# On Feature Learning in
# Structured State Space Models

**Leena Chennuru Vankadara**[†1]     **Jin Xu**[†2]     **Moritz Haas**[3]     **Volkan Cevher**[1,4]

[1]AGI Foundations, Amazon                    [2]University of Oxford[*]
[3]University of Tübingen, Tübingen AI Center,[*]    [4]LIONS, EPFL[*]

## Abstract

This paper studies the scaling behavior of state-space models (SSMs) and their structured variants, such as Mamba, that have recently arisen in popularity as alternatives to transformer-based neural network architectures. Specifically, we focus on the capability of SSMs to learn features as their network width approaches infinity. Our findings reveal that established scaling rules, such as the Maximal Update Parameterization, fail to support feature learning as these models cannot be represented in the form of Tensor Programs. Additionally, we demonstrate that spectral scaling conditions, shown to be effective for feature learning in a host of other architectures, do not hold the same implications for SSMs. Through a detailed signal propagation analysis in SSMs, both forward and backward, we identify the appropriate scaling necessary for non-trivial feature evolution in the infinite-width limit. Our proposed scaling shows behavior akin to the Maximal Update Parameterization, such as improved stability, better generalization, and transferability of optimal hyper-parameters from small to large scale SSMs.

## 1   Introduction

State-space models (SSMs), such as Mamba (Gu and Dao, 2023), have become popular in deep learning as alternatives to transformers like GPT and BERT series (Radford et al., 2019, Brown et al., 2020, Achiam et al., 2023, Devlin et al., 2018, Touvron et al., 2023, Chowdhery et al., 2023, Gemini Team et al., 2023). SSMs integrate elements from RNNs, CNNs, and control models, excelling in inference and handling long contexts (Gu et al., 2021, Gupta et al., 2022, Gu et al., 2022, Smith et al., 2022).

The success of foundation models based on transformers and SSMs alike is largely attributed to their scale—both in terms of data and model size. However, this increased scale often introduces challenges, such as precision issues due to instability or the vanishing/exploding gradient problems. Additionally, the sequential nature of state-space models (SSMs) makes them notoriously difficult to train. Therefore, developing a rigorous understanding of how SSMs scale as their dimensions increase and identifying optimal scaling rules is crucial.

In this vein, infinite-width asymptotics, such as Neural tangent kernel (NTK) analyses have been a central tool providing key insights in DL theory. However, a key limitation of the NTK analysis is the lack of *feature learning* in the infinite width limit (Yang and Littwin, 2023). Feature learning addresses the more realistic training setting where we have unrestricted movement of the neural network parameters (Yang and Littwin, 2023).

Intriguingly, recent work by Yang and Hu (2021) showed that under an expanded space of hyper-parameters, which includes layer-wise scaling of the learning rates, one can find a unique parameterization called Maximal Update pPrameterization ($\mu$P) that induces non-trivial feature evolution

---

[*]† denotes equal contribution. This work was conducted during Jin's, Moritz' and Volkan's time at Amazon. Correspondence to: aaron.jin.xu@gmail.com

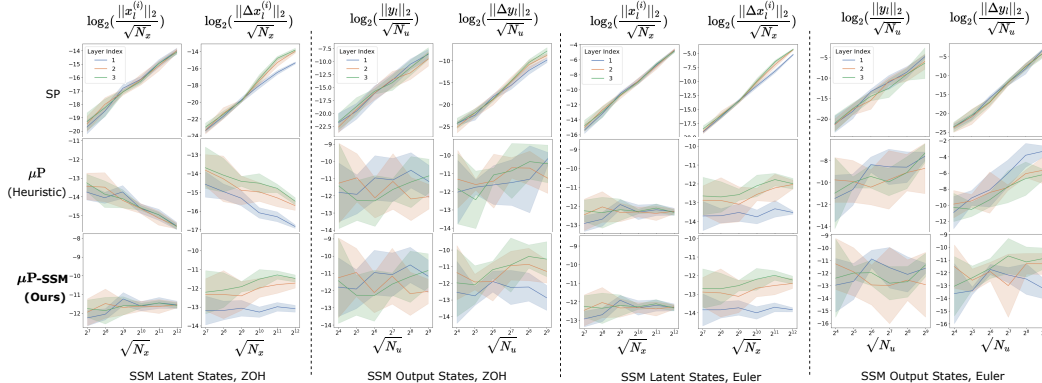

Figure 1: Under our derived scaling $\mu$P-SSM, Mamba achieves feature learning in all three SSM layers. In contrast, both Standard Parametrization (SP) and $\mu$P (heuristic) lead to instability or vanishing updates for either the latent states $\mathbf{x}_l^{(i)}$ or the output signal $\mathbf{y}_l$ or both in each SSM layer. The figures above illustrate the scalings when $l = 1$, but they exhibit the same trend across recurrence steps. We simultaneously scale up $N_x$ and $N_u$. We run each experiment 10 times, and the shaded areas indicate the standard deviation of these runs. Both Zero-Order Hold (ZOH) and Euler discretization of $\mathbf{B}_l$ are studied and indicated in the subtitle.

even in the infinite width limit for a host of key architectures. The results are obtained via the mathematical framework of Tensor Programs, which provides the foundation to study the effects of parameterizations on the learning dynamics in the limit. To this end, it is natural to ask:

*When do SSMs admit feature learning in the infinite-width limit?*

Our research tackles this question by investigating the behavior of SSMs as network width approaches infinity. We examine how SSMs learn and evolve features in this context and assess the adequacy of recent scaling rules such as $\mu$P and spectral scaling conditions.

**Our contributions:**

- We provide a detailed scaling analysis of forward and backward signal propagation in SSMs as the width approaches infinity, identifying that standard scalings lead to unbounded signals.

- We demonstrate theoretically and empirically that popular scaling rules like $\mu$P do not yield correct scaling for SSMs due to—as we prove—their non-representability as Tensor Programs (*cf.*, Figure 1).

- We derive a unique correction that ensures correctly balanced signals in both the forward and backward passes, stabilizing the training process and enhancing model performance.

- Empirically, we validate that our proposed scaling facilitates hyper-parameter transfer from small-scale to large-scale SSMs, similar to the effects observed in MLPs and transformers.

## 2 On the lack of feature learning in SSMs at infinite width

### 2.1 Feature learning in sequence models

Sequence models are built by combining sequential layers that transform input sequences into output sequences. A sequence layer can be represented as $\mathbf{y}_{1:L} = f_{\text{seq}}(\mathbf{u}_{1:L})$ where $\mathbf{u}_{1:L}$ is a compact notation for sequence $\mathbf{u}_1, \ldots, \mathbf{u}_L$, with $\mathbf{u}_l \in \mathbb{R}^{N_u}$, $\mathbf{y}_l \in \mathbb{R}^{N_y}$. Note that $f_{\text{seq}}$ can be easily generalized to accept and output multiple sequences as in residual connections and multi-branch architectures (see Appendix B.1). Here $N_u$ and $N_y$ are treated as widths of the sequence model, which can be scaled up to create larger sequence models. To denote the backward pass of a model concisely, let $\mathcal{L}$ be the overall training loss, and we write $\bar{\bullet} = \frac{\partial \mathcal{L}}{\partial \bullet}$. Instead of studying the scaling of each element in a vector $\mathbf{u} \in \mathbb{R}^{N_u}$, we study the norm $\|\mathbf{u}\|_2$. When elements in $\mathbf{u}$ are in $\Theta(1)$, we have $\|\mathbf{u}\|_2 \in \Theta(\sqrt{N_u})$. Throughout this work, we will use $\Theta$ to represent the asymptotic order in the limit ($N_u \to \infty$ in this case), but omit the precise notation for readability.

In layerwise sequence models, where the $k$-th layer is denoted as $\mathbf{u}_{1:n}^{(k)} = f_k(\mathbf{u}_{1:n}^{(k-1)})$ with $\mathbf{u}_l^{(k)} \in \mathbb{R}^{N_k}$, we have the following definition for feature learning:

**Definition 2.1** (Feature Learning in Layerwise Sequence Models). A layerwise sequence model is in the *feature learning regime* if for any $k \in [K]$, the features $\mathbf{u}_{1:L}^{(k)}$ and the updates of features $\Delta\mathbf{u}_{1:L}^{(k)}$ after one gradient update have the following scaling:

$$\exists\, l \in [L],\ \|\mathbf{u}_l^{(k)}\|_2 \in \Theta(\sqrt{N_k}) \quad \text{(Stability at initialization)} \tag{INIT}$$

$$\exists\, l \in [L],\ \|\Delta\mathbf{u}_l^{(k)}\|_2 \in \Theta(\sqrt{N_k}) \quad \text{(Non-trivial feature updates)} \tag{$\Delta$}$$

Note that the first condition is a stability condition that demands the activations to remain coordinate-wise $\Theta(1)$ in the forward pass, whereas the second condition ensures nonvanishing and nonexploding activation updates. This definition can be easily generalized to general cases for arbitrary sequence models in Appendix B.1.

For a particular sequence layer in the model, if we assume the inputs to the layer are asymptotically independent and identically distributed (i.i.d), correctly scaled, and the gradients backpropagated into the sequence layer have the correct scaling (as in Assumption 3.1), and this sequence layer satisfies the two conditions (INIT) and ($\Delta$), we say this sequence layer admits feature learning. The corrected scaling $\mu P$-SSM we propose in Section 3 satisfies an even stronger condition: The updates of *all trainable weights* should have nonvanishing and nonexploding effect on the activation and output function updates. We then say that the sequence model is *effectively feature learning*. This is similar in spirit to the requirements for $\mu P$ from Yang and Hu (2021) that ensure maximal stable updates of all trainable weights in standard architectures like MLPs, but requires different initialization and learning rate scaling rules for the trainable weights in SSMs, as we show in Section 3.

## 2.2 Tensor Programs, spectral scaling, and the Maximal Update Parmeterization

**Tensor Programs.** The framework of Tensor Programs (TP) (Yang, 2019) was initially developed to understand the behavior of wide neural networks both at initialization and during training. While there are many versions of Tensor Programs, the most general version (that subsumes all the previous versions) is referred to as the NE⊗OR⊤ program (Yang and Littwin, 2023). A NE⊗OR⊤ program constitutes a sequence of vectors in $\mathbb{R}^n$ and a sequence of scalars in $\mathbb{R}$ inductively generated from an initial set of random scalars, vectors, and matrices following a specified set of instructions: matrix multiplications, non-linear outer products, and vector averages. Yang and Littwin (2023) show that for any architecture whose forward pass can be represented as a NE⊗OR⊤ program (which includes many modern architectures used in practice including transformers and convolutional networks), there exists a unique scaling rule called Maximal Update Parametrization ($\mu P$) under which features in every single layer evolve in a width-independent fashion with scale. Interestingly, it has been shown that unlike standard parameterizations, $\mu P$ allows transferability of optimal hyper-parameters from small to large scale models.

The key idea of the "Tensor Program Ansatz" (Yang, 2019) is that pre-activations and their gradients arising in training most neural network architectures via standard update rules (such as SGD and ADAM) can be represented as vectors in a NE⊗OR⊤ program. The Ansatz suggests that vectors in a TP become asymptotically independent as well as identically distributed both at initialization as well as during training. Leveraging this, the framework allows for a mechanistic tracking of the behaviour of such vectors by assigning a random variable $Z^v$ to represent the (asymptotically identical) distribution of the coordinates of the vector $v$. This principle underlies the development of the key theoretical result of the Tensor Program machinery — the Master Theorem — which can be viewed as the compositional, non-linear generalization of the law of large numbers. It allows the theoretical computation of infinite width limits of different quantities such as (pre-)activations or the output of a neural network.

While the following proposition also holds for other structured SSMs such as MAMBA, for clarity, we first introduce the simpler S4 model.

**The S4 recurrent layer.** The S4 recurrent layer $\mathbf{y}_{1:L} = f_{\text{S4}}(\mathbf{u}_{1:L}; \mathbf{w} = \{\mathbf{B}, \mathbf{C}\})$ can be viewed as a discretization of a continuous-time SSM given by

$$\frac{d}{dt}\mathbf{x}_t = \mathbf{A}\mathbf{x}_t + \mathbf{B}\mathbf{u}_t \quad \text{and} \quad \mathbf{y}_t = \text{Re}[\mathbf{C}\mathbf{x}_t] \tag{1}$$

for $l = 1, \ldots, L$ with $\mathbf{x}_0 = 0$ and where $\mathrm{Re}(\cdot)$ gives the real part of a complex vector, and where we let $\mathbf{A} = \mathrm{Diag}(\mathbf{a})$ with $\mathbf{a} \in \mathbb{C}^{N_x}$, $\mathbf{B} \in \mathbb{C}^{N_x \times N_u}$, $\mathbf{C} \in \mathbb{C}^{N_y \times N_x}$. We have the discretized sequence to sequence mapping $f_{\mathrm{S4}}$ as follows:

$$\mathbf{x}_l = \mathbf{A}' \mathbf{x}_{l-1} + \mathbf{B}' \mathbf{u}_l \quad \text{and} \quad \mathbf{y}_l = \mathrm{Re}[\mathbf{C}' \mathbf{x}_l] \tag{2}$$

where we follow the Zero-Order-Hold (ZOH) discretization method,

$$\mathbf{A}' = \exp(\tau \cdot \mathbf{A}), \quad \mathbf{B}' = (\mathbf{A}' - \mathbf{I}) \mathbf{A}^{-1} \mathbf{B}, \quad \text{and} \quad \mathbf{C}' = \mathbf{C}. \tag{3}$$

In practice, simple Euler discretization can also be used for $\mathbf{B}$, i.e., $\mathbf{B}' = \tau \mathbf{B}$. To model long-range dependency, S4 advocates for HiPPO theory (Gu et al., 2020). There are many possible Hippo-based initializations possible for the structured transition matrices $\mathbf{A}$ (e.g., Hippo-Leg-S, S4D-Inv, and S4D-Real) and our result holds for all the parameterizations.

While the training of most common architectures including transformers or convolutional networks via update rules such as SGD or ADAM are representable as a NE⊗OR⊤ program, Proposition 2.2 shows that this does not hold in general for structured state space models.

**Proposition 2.2 (Structured SSMs are not generally representable as NE⊗OR⊤ programs).** *The forward and backward signal propagation of structured SSMs including S4 (2) and (3) and MAMBA (Section 3.1) trained via standard algorithms such as SGD or ADAM are not representable as a NE⊗OR⊤ program. Indeed this holds for all existing Hippo-parameterizations of the structured matrix $\mathbf{A}$ including HiPPO-LegS, HiPPO-LegS-N, HiPPO-LegS-D, S4D-Inv, S4D-Lin, and S4D-Real.*

*Proof sketch.* The formal proofs are provided in Appendix C. The key architectural component that is not representable in a NE⊗OR⊤ program is the Hippo-based structured transition matrix $\mathbf{A}$. Due to the rapid decay of the diagonal entries of $\mathbf{A}^{-1}$, the hidden states $\mathbf{x}_0$ are not even asymptotically identically distributed, already at initialization. Therefore coordinates of $\mathbf{x}_0$ cannot be generated by a sequence of NE⊗OR⊤ computations, neither at initialization nor over the course of training. As we will discuss in Section 6, developing a TP-like framework to cover SSMs requires a substantial generalization of the existing TP framework and is beyond the scope of the current paper. Nevertheless, in Section 6, we outline the key challenges and potential paths toward such a generalization.

**Spectral scaling condition.** Yang et al. (2023a) showed that many practical architectures admit feature learning if certain spectral scaling conditions are satisfied. Let $\mathbf{W}_l$ and $\Delta \mathbf{W}_l \in \mathbb{R}^{N_l \times N_{l-1}}$ denote the $l$th layer weight matrices and their gradient updates. The condition states that for feature learning to hold, the spectral norms of the matrices should satisfy:

$$\|\mathbf{W}_l\|_* \in \Theta\left(\sqrt{\frac{N_l}{N_{l-1}}}\right) \quad \text{and} \quad \|\mathbf{\Delta W_l}\|_* \in \Theta\left(\sqrt{\frac{N_l}{N_{l-1}}}\right) \quad \text{for all } l \in [L]. \tag{*}$$

Under $\mu$P, architectures that are representable as a NE⊗OR⊤ program satisfy the spectral scaling condition (Yang et al., 2023a). Here, we show that for structured SSMs, architectures that satisfy spectral scaling conditions do not in general satisfy conditions for feature learning.

**Proposition 2.3 (Spectral scaling does not generally imply feature learning in SSMs).** *Structured SSMs including S4 and Mamba trained via standard algorithms such as SGD or ADAM that satisfy spectral scaling conditions (∗) do not satisfy conditions for feature learning given in condition (Δ). Indeed this holds for all well-known Hippo-parameterizations of the structured matrix $\mathbf{A}$ including HiPPO-LegS, HiPPO-LegS-N, HiPPO-LegS-D, S4D-Inv, S4D-Lin, and S4D-Real.*

*Proof sketch.* The formal proofs are provided in Appendix C. To gather some intuition for this result in a simplified setting, consider the scale of the hidden states for token index 0, which is already wrong at initialization. By definition, $\|\mathbf{x}_1\|_2 = \|\mathbf{B}' \mathbf{u}_1\|_2 = \|\mathbf{\Lambda B u}_1\|_2$, where $\mathbf{\Lambda} = (\mathbf{A}' - \mathbf{I}) \mathbf{A}^{-1}$. Now, observe that $\|\mathbf{B}' \mathbf{u}_1\|_2^2$ is a sum of independent random variables that satisfy the Kolmogorov condition, so that the sum behaves according to the strong law of large numbers. For spectral scaling conditions to yield the right scaling of the initialization variance, it is crucial that the following condition holds:

$$\|\mathbf{B}' \mathbf{u}_1\|_2 \in \Theta\left(\|\mathbf{B}'\|_* \|\mathbf{u}_1\|_2\right). \tag{4}$$

However, using standard tools from random matrix theory one can show that $\frac{\|\mathbf{B}' \mathbf{u}\|_2}{(\|\mathbf{B}'\|_* \|\mathbf{u}_1\|_2)} \in \Theta(\frac{1}{\sqrt{N_u}})$ which clearly violates (4).

| | ZOH Discretization $\mid$ Euler Discretization | | |
| --- | --- | --- | --- |
| | SP | $\mu$P (Heuristic) | $\mu$P-SSM (Ours) |
| $\sigma_B$ | $\frac{1}{\sqrt{N_u}}$ | $\frac{1}{\sqrt{N_u}} \min\{1, \sqrt{\frac{N_x}{N_u}}\}$ | $\sqrt{\frac{N_x}{N_u}} \mid \frac{1}{\sqrt{N_u}}$ |
| $\sigma_C$ | $\frac{1}{\sqrt{N_u}}$ | $\frac{1}{\sqrt{N_u}} \min\{1, \sqrt{\frac{N_x}{N_u}}\}$ | $\frac{1}{\sqrt{N_x N_u}}$ |
| $\eta_a$ | $1$ | $\sqrt{\frac{N_u}{N_x}}$ | $N_u \mid \sqrt{N_x} N_u$ |
| $\eta_B$ | $1$ | $\frac{N_x}{N_u}$ | $\frac{N_x}{\sqrt{N_u}} \mid \sqrt{\frac{N_x}{N_u}}$ |
| $\eta_C$ | $1$ | $\frac{N_x}{N_u}$ | $\frac{1}{N_x \sqrt{N_u}}$ |
| $\left\|\mathbf{x}_l^{(i)}\right\|_2$ | $\Theta(1) \mid \Theta(\sqrt{N_x})$ | $\Theta(\min\{1, \sqrt{\frac{N_x}{N_u}}\}) \mid \Theta(\sqrt{N_x} \min\{1, \sqrt{\frac{N_x}{N_u}}\})$ | $\Theta(\sqrt{N_x})$ |
| $\|\mathbf{y}_l\|_2$ | $\Theta(\sqrt{N_u}) \mid \Theta(\sqrt{N_x N_u})$ | $\Theta(\sqrt{N_u} \min\{1, \frac{N_x}{N_u}\}) \mid \Theta(\sqrt{N_x N_u} \min\{1, \frac{N_x}{N_u}\})$ | $\Theta(\sqrt{N_u})$ |
| $\left\|\Delta\mathbf{x}_l^{(i)}\right\|_2$ | $\Theta(\sqrt{N_u}) \mid \Theta(\sqrt{N_x N_u})$ | $\Theta(\frac{N_x}{\sqrt{N_u}} \min\{1, \sqrt{\frac{N_x}{N_u}}\}) \mid \Theta(\frac{N_x \sqrt{N_x}}{\sqrt{N_u}} \min\{1, \sqrt{\frac{N_x}{N_u}}\})$ | $\Theta(\sqrt{N_x})$ |
| $\|\Delta\mathbf{y}_l\|_2$ | $\Theta(N_u) \mid \Theta(\sqrt{N_x} N_u)$ | $\Theta(\frac{N_x}{\sqrt{N_u}} \min\{1, \frac{N_x}{N_u}\}) \mid \Theta(\frac{N_x \sqrt{N_x}}{\sqrt{N_u}} \min\{1, \frac{N_x}{N_u}\})$ | $\Theta(\sqrt{N_u})$ |

Table 1: Overview of the different parameterizations and their corresponding scaling for latent states, outputs and their updates. Results for ZOH and Euler discretization are separated by |.

## 3 Identifying the unique scaling for effective feature learning in SSMs

In this section, we analyze the forward and backward signal propagation in structured SSMs. Due to the generality of the architecture, we consider MAMBA as the basis for the analysis. Similar arguments apply for other SSMs such as S4, S5, H3, or DSS. In the appendix, we also provide a detailed analysis of signal propagation in S4 and identify the correct scaling conditions for maximal stable weight updates.

### 3.1 Selective State Space Models

Selective SSMs in the Mamba architecture $\mathbf{y}_{1:L} = f_{\text{Mamba}}(\mathbf{u}_{1:L})$ can be written as follows:

$$\mathbf{B}_l = \text{Lin}_{N_x}(\mathbf{u}_l; \mathbf{W}_B, \mathbf{b}_B) \in \mathbb{R}^{N_x}, \tag{5}$$

$$\mathbf{C}_l = \text{Lin}_{N_y}(\mathbf{u}_l; \mathbf{W}_C, \mathbf{b}_C) \in \mathbb{R}^{N_x}, \tag{6}$$

$$\tau_l = \text{Softplus}(\tau_0 + \text{Broadcast}_{N_u}(\text{Lin}_1(\mathbf{u}_l; \mathbf{W}_\tau, \mathbf{b}_\tau)), \ \tau_0, \tau_l \in \mathbb{R}^{N_u}. \tag{7}$$

For $i = 1, \ldots, N_u$, we have $N_u$ 1D SSMs:

$$\mathbf{A}^{(i)} = \text{Diag}(-\exp(\mathbf{a}_{\log}^{(i)})), \ \mathbf{a}_{\log}^{(i)} \in \mathbb{R}^{N_x}, \quad \mathbf{x}_l^{(i)} = \mathbf{A}_l^{\prime(i)} \mathbf{x}_{l-1}^{(i)} + u_l^{(i)} \mathbf{B}_l^{\prime(i)}, \ u_l^{(i)} \in \mathbb{R}, \ \mathbf{x}_l^{(i)} \in \mathbb{R}^{N_x},$$

$$\mathbf{A}_l^{\prime(i)}, \mathbf{B}_l^{\prime(i)} = \text{ZOH}(\tau_l^{(i)}, \mathbf{A}^{(i)}, \mathbf{B}_l), \ \tau_l^{(i)} \in \mathbb{R}, \qquad y_l^{(i)} = \mathbf{C}_l^{\mathsf{T}} \mathbf{x}_l^{(i)}, \ y_l^{(i)} \in \mathbb{R}.$$

where $\mathbf{x}_0^{(i)} = \mathbf{0}$, $\mathbf{a}_{\log}^{(i)}$, $\tau_0 \in \mathbb{R}^{N_u}$, and all linear layer weights $\mathbf{W}_B, \mathbf{b}_B, \mathbf{W}_C, \mathbf{b}_C, \mathbf{W}_\tau, \mathbf{b}_\tau$ are trainable parameters. The Zero-Order-Hold (ZOH) discretization procedure $\mathbf{A}_l^{\prime(i)}, \mathbf{B}_l^{\prime(i)} = \text{ZOH}(\tau, \mathbf{A}^{(i)}, \mathbf{B}_l^{\prime(i)})$ can be written as:

$$\mathbf{A}_l^{\prime(i)} = \exp(\tau_l^{(i)} \cdot \mathbf{A}^{(i)}), \qquad \mathbf{B}_l^{\prime(i)} = (\mathbf{A}_l^{\prime(i)} - \mathbf{I})\mathbf{A}^{(i)-1}\mathbf{B}_l^{(i)}. \tag{8}$$

In a nutshell, Mamba can be seen as $N_u$ SSMs, one for each input channel. Weights for these SSMs are shared and depend on the input at that recurrent step. Because of the dependency on inputs, Mamba can model non-stationary sequences. Weight matrices are initialized as follows:

$$\mathbf{W}_B \sim \mathcal{N}(\mathbf{0}, \sigma_B^2 \mathbf{I}) \in \mathbb{R}^{N_x \times N_u}, \quad \mathbf{W}_C \sim \mathcal{N}(\mathbf{0}, \sigma_C^2 \mathbf{I}) \in \mathbb{R}^{N_x \times N_u}, \quad \mathbf{W}_\tau \sim \mathcal{N}(\mathbf{0}, \sigma_\tau^2 \mathbf{I}) \in \mathbb{R}^{1 \times N_u},$$

and zero initialization for all biases. For each SSM, $\mathbf{A}^{(i)}$ is still initialized according to the HiPPO theory. When $\mathbf{A}^{(i)}$ is real-valued, we let $\mathbf{a}_{\log}^{(i)}[j] = \log(j + 1)$ and $\tau_0$ can be seen as a bias term initialized to $\tau_0 \sim \text{Softplus}^{-1}(\mathcal{U}(0.001, 0.1))$. We will assume that $\tau_0$ is not trained, as this does not have any effect on the scale of the different quantities under consideration. This is a minor technical assumption and our results would also hold if we considered $\tau$ as a trainable parameter.

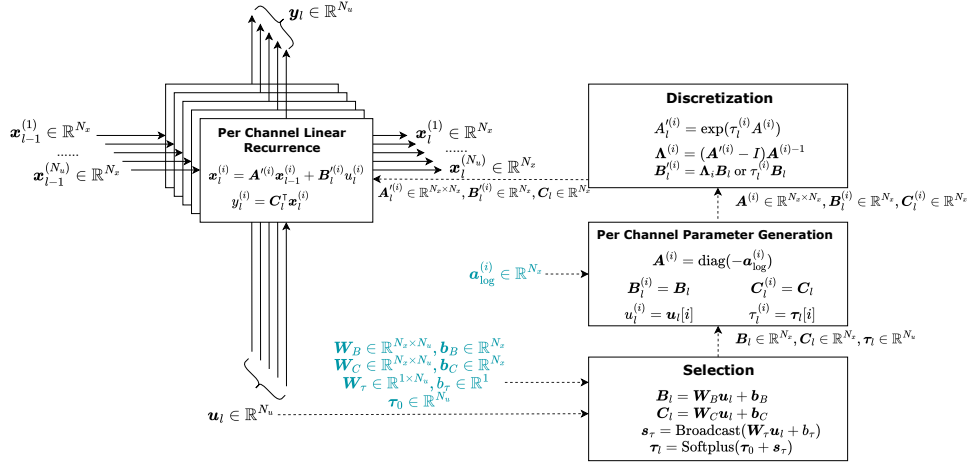

Figure 2: **Illustration of the Mamba S6 Layer**. The computation is modularized into three components: selection, discretization, and per-channel linear recurrence. Mamba introduces a selection mechanism where weight matrices $\mathbf{B}_l$, $\mathbf{C}_l$ depend on the inputs $\mathbf{u}_l$. These weight matrices are then separated into per-channel parameters, and discretized using either the ZOH or Euler methods. The discretized, per-channel weights are then applied in a linear recurrence, allowing each channel to perform computations in parallel. Trainable parameters are shown in blue.

## 3.2 Forward signal propagation through a S6 Mamba layer

To derive the correct choice of initialization scalings $\sigma_B$ and $\sigma_C$, we begin by analyzing the scale of activations (i.e., hidden states and outputs) in a S6 Mamba layer in the first forward pass as $N_u \to \infty$ then $N_x \to \infty$. We believe that our results would also hold in the proportional limit where $N_u \to \infty$ and $N_x \to \infty$ with $\frac{N_x}{N_u} \in \Theta(1)$. However deriving the results in this setting would incur a significant technical overhead and we defer this analysis to a future work. We briefly discuss this in Section 6. As is common in practice, we assume that the SSM layer is embedded into a neural network architecture containing standard architectural blocks such as MLPs, normalization layers, or residual connections. All proofs are provided in Appendix C.2.

**Assumption 3.1.** *Assume that the forward pass of all the components of the network except the SSM layer are expressible as* $\textsc{Ne}\otimes\textsc{or}\top$ *programs and are parameterized according to $\mu P$.*

This assumption ensures that the inputs to the SSM layer are asymptotically i.i.d and correctly scaled. It also ensures that gradients into the SSM layer have the correct scaling. All results in this section are stated under Assumption 3.1. Through the forward signal propagation analysis, we identify the correct scale of initialization for weight matrices $\mathbf{W}_B$ and $\mathbf{W}_C$. We show that both standard parameterization as well as spectral scaling conditions do not yield the correct scale of initialization for the weight matrices. The key results are summarized in Table 1.

For simplicity of exposition, first consider the scale of $\mathbf{x}_1^{(i)} = (\mathbf{A}_l'^{(i)} - \mathbf{I})\mathbf{A}^{(i)-1}\mathbf{B}_1 u_1^{(i)}$ before generalizing to arbitrary $l \in [L]$.

**Proposition 3.2 (Scale of hidden states $\mathrm{x}_1^{(i)}$ in Mamba at initialization).** *Under the ZOH discretization procedure, as $N_u$ then $N_x$ approach infinity, for any $i \in [N_u]$, the squared $l_2$-norm of the hidden states $\|\mathbf{x}_1^{(i)}\|_2^2$ is a.s. scaled as $\|\mathbf{x}_1^{(i)}\|_2^2 \in \Theta\left(\zeta(2)\,\sigma_B^2\,\|\mathbf{u}_1\|^2\,(u_1^{(i)})^2\right)$, where $\zeta(2)$ denotes the Riemann zeta function at 2.*

Following condition (INIT), the 1D SSM admits stability at initialization under the following conditions:

$$\text{If } |u_l^{(i)}| \in \Theta(1) \text{ then } \|\mathbf{x}_l^{(i)}\| \in \Theta(\sqrt{N_x}) \text{ and } |y_l^{(i)}| \in \Theta(1).$$

Therefore, for stability at initialization, the initialization should scale as $\sigma_B \in \Theta(\sqrt{\frac{N_x}{N_u}})$. Note that, under both standard parameterization (e.g., Kaiming or LeCun initialization) and spectral scaling conditions, $\sigma_B$ is initialized as $\Theta(\sqrt{\frac{1}{N_u}})$, which leads to vanishing hidden states according to Proposition 3.2. We empirically verify this fact in Figure 1.

Proposition 3.3 provides the scale of the output of a Mamba layer.

**Proposition 3.3 (Scale of outputs $\mathbf{y}_1^{(i)}$ of a Mamba layer at initialization).** *Under the ZOH discretization procedure as $N_u$ then $N_x$ approach infinity, for any $i \in [N_u]$, the output $y_1^{(i)}$ converges in distribution to a Gaussian with mean $0$ and standard deviation $C\sigma_B\sigma_C \|\mathbf{u}_1\|_2^2$ for some width-independent constant $C > 0$.*

Accordingly, imposing the conditions for stability of $y_1^{(i)}$ according to condition (INIT) implies the initialization scaling condition $\sigma_C \in \Theta(\sqrt{\frac{1}{N_x N_u}})$. Note again that standard parameterization suggests initializing $\sigma_C \in \Theta(\sqrt{\frac{1}{N_u}})$ under which the outputs would diverge with scale. Under spectral scaling, $\sigma_C$ is initialized much larger. However, since the hidden states vanish with width, the outputs of the SSM admit the correct scaling here as demonstrated in Figure 1.

**Generalizing to arbitrary $l \in [L]$.** The corrected scalings of $\sigma_B$ and $\sigma_C$ derived above generalize to the entire sequence, as a sum over the sequence usually does not cancel out the scaling. More formally, for all $l \in [L]$, we have

$$\mathbf{x}_l^{(i)} = \sum_{m=0}^{l-1} (\mathbf{A}_l'^{(i)})^m \mathbf{B}_{l-m}'^{(i)} u_{l-m}^{(i)}.$$

First, observe that the operator $(\mathbf{A}_l'^{(i)})^m$ does not change the width-scaling. To see this note that since $\mathbf{A}_l'^{(i)} = \mathrm{Diag}(a_1', \ldots, a_{N_x}')$ with $a_n' = e^{-\frac{1}{2}\tau_l^{(i)}}(\cos(\tau_l^{(i)}\pi n) + i\sin(\tau_l^{(i)}\pi n))$, we have that, for all complex vectors $\mathbf{v} \in \mathbb{C}^{N_x}$ it holds that $\|(\mathbf{A}_l'^{(i)})^m \mathbf{v}\|_2 = e^{-m\tau_l^{(i)}/2}\|\mathbf{v}\|_2$ for any $m \in [L]$. Now since, for any $\mathbf{u}_l$, setting $\sigma_B \in \Theta(\sqrt{\frac{N_x}{N_u}})$ yields $\left\|\mathbf{B}_l'^{(i)}\mathbf{u}_l\right\|_2 \in \Theta(\sqrt{N_x})$, each term in the summation is of order $\Theta(\sqrt{N_x})$. Unless, for every $l$, the term $\mathbf{B}_l'^{(i)}\mathbf{u}_l$ perfectly cancels out with the terms before to affect the width scaling , we have that $\left\|\mathbf{x}_l^{(i)}\right\|_2 \in \Theta(\sqrt{N_x})$. The same argument can be used to show the stability of $y_l^{(i)}$. Concluding this argument, we have derived the correct scaling of the initialization variances $\sigma_B$ and $\sigma_C$ for feature stability in a Mamba layer summarized below.

> **Conditions for stability of a S6 Mamba layer at initialization.** The features of a S6 Mamba recurrent layer $\mathbf{y}_{1:L} = f_{\mathrm{mamba}}(\mathbf{u}_{1:L}; \mathbf{w})$ are stable at initialization in the infinite-width limit under the following scaling conditions:
>
> $$\sigma_B \in \Theta\left(\sqrt{\frac{N_x}{N_u}}\right) \quad \text{and} \quad \sigma_C \in \Theta\left(\sqrt{\frac{1}{N_x N_u}}\right)$$

### 3.3 Backward signal propagation in a S6 Mamba Layer

In this section, we provide the correct scaling of the learning rates $\eta_a$, $\eta_B$, and $\eta_C$ by a detailed analysis of the backward signal propagation in the limit of $N_u \to \infty$ then $N_x \to \infty$. Specifically, we analyze the scale of the activation updates for both hidden states and outputs in the first backward pass through a Mamba layer. We also show that both standard parameterization as well as spectral scaling conditions do not yield the correct scale of learning rates for the weight matrices. The key results are summarized in Table 1.

**Proposition 3.4 (Scale of the updates of hidden states $\Delta\mathbf{x}_1^{(i)}$ after 1 step of SGD).** *Under the ZOH discretization procedure, as $N_x$ then $N_u$ approach infinity, for every 1D SSM, the squared $l_2$-norm of the updates $\|\Delta\mathbf{x}_1^{(i)}\|_2$ of the hidden states after one step of SGD is a.s. scaled as $\|\Delta\mathbf{x}_1^{(i)}\|_2 \in \Theta\left(\eta_B \frac{1}{\sqrt{N_u}}\sigma_C \|\mathbf{u}_1\|_2^3 \zeta(4)^{\frac{1}{2}}\right)$, where $\zeta(4)$ denotes the Riemann zeta function at 4.*

The Riemann zeta function at 4 evaluates to a width-independent constant and $\sigma_C \in \Theta(\sqrt{\frac{1}{N_x N_u}})$ due to Proposition 3.3 for stability of outputs. Therefore, for the scale of the hidden state updates to be $\Theta(\sqrt{N_x})$, the correct scaling of the learning rate $\eta_B$ is given by $\Theta(\frac{N_x}{\sqrt{N_u}})$. On the other hand, spectral scaling suggests that $\eta_B$ must scale as $\Theta(\frac{N_x}{N_u})$. Under this scaling, however, updates of the hidden states would vanish with width and therefore the first block of the SSM is in the *lazy regime*. This is corroborated by our experiments in Figure 1.

Next, to derive the correct scaling of the learning rate $\eta_C$, we consider the scale of output updates.

**Proposition 3.5** (**Scale of the updates of outputs $\Delta \mathbf{y}_1^{(i)}$ after 1 step of SGD**). *Under the ZOH discretization procedure, as $N_u$ then $N_x$ approach infinity, for every 1D SSM, the squared $l_2$-norm of the updates of the hidden states after one step of SGD scales as* $|\Delta y_1^{(i)}|^2 \in \Theta\left(\eta_C \sigma_B^2 \sqrt{\frac{1}{N_u}} \|\mathbf{u}_1\|_2^4\right)$.

The result suggests that for the correct scaling of the updates, the learning rate $\eta_C$ must scale as $\Theta(\frac{1}{N_x \sqrt{N_u}})$. Under spectral scaling, $\eta_C$ scales much larger as $\Theta(\frac{N_x}{N_u})$. However, since the updates of the hidden states vanish under spectral scaling, this larger incorrect scaling of $\eta_C$ downward corrects the scale of the updates in the outputs as shown in Table 1 and empirically verified in Figure 1.

**On the correct scaling of $\eta_a$.** It turns out that the scaling of the learning rate $\eta_a$ does not play a role in either stability at initialization or for non-trivial updates with scale. However, if $\eta_a$ is not scaled correctly, then the transition matrix $\mathbf{A}$ is not updated. In particular, as shown in Appendix C.2, $\eta_a$ needs to scale as $\Theta(N_u)$. Below we summarize the correct scaling conditions to achieve non-trivial feature updates in the infinite-width limit of a Mamba layer.

In the same vein as the discussion in Section 3.2, it is straightforward to verify that our results hold for arbitrary $l \in [L]$ and more gradient steps as soon as we assume that the updates of the weight matrices and activations do not perfectly cancel out the corresponding initial quantities.

---

**Conditions for non-trivial feature updates in a S6 Mamba Layer.** The updates in a S6 Mamba recurrent layer $\mathbf{y}_{1:L} = f_{\text{mamba}}(\mathbf{u}_{1:L}; \mathbf{w})$ evolve non-trivially in the infinite-width limit under the following conditions:

$$\sigma_B \in \Theta(\sqrt{\frac{N_x}{N_u}}), \quad \sigma_C \in \Theta(\frac{1}{\sqrt{N_x N_u}}), \quad \eta_a \in \Theta(N_u), \quad \eta_B \in \Theta(\frac{N_x}{\sqrt{N_u}}), \text{ and } \eta_C \in \Theta(\frac{1}{N_x \sqrt{N_u}})$$

---

# 4 $\mu$P-SSM implies stability and feature learning in Mamba

**Empirical verification of different scalings in Table 1.** First, we verify that our derived $\mu$P-SSM scaling (see Table 1) for Mamba indeed leads to feature learning, i.e., it ensures stability at initialization as defined in condition (INIT) and non-trivial feature updates during training as defined in condition ($\Delta$). We scale up the SSM latent state size $N_x$ and the SSM output dimension $N_u$ simultaneously and track the scaling of both features and feature updates. Due to the linear decay in the eigenvalues of the transition matrix $\mathbf{A}^{-1}$, we typically observe a strong finite sample effect at small $N_x$. Constrained by computational resources, we opt for a much smaller $N_u$ ($N_u = N_x/8$) than is usually employed in practice. This adjustment enables us to scale up $N_x$ effectively, thus mitigating the finite-sample effect and to more clearly demonstrate the scaling behavior in the asymptotic limit in Figure 1. For all experiments in this section, we train Mamba with 3 SSM blocks for language modelling on the wikitext dataset (Merity et al., 2016) and use plain Stochastic Gradient Descent (SGD) to perform gradient updates. We use the huggingface (Wolf et al., 2019) Mamba implementation and the $\mu$P package (Yang et al., 2022) for scaling in our experiments.

As shown in Figure 1, under Standard Parametrization (SP), both the SSM latent states and the outputs explode at initialization, and their updates also explode, leading to instability both at initialization and during training. Under the spectral scaling parameterization prescribed in Yang et al. (2023a), other layers except for the SSM layers have the correct scaling by design. However, in the SSM layer, when using Zero-Order Hold (ZOH) discretization for $\mathbf{B}_l$, the latent states at initialization and their updates vanish when scaling up the width, while the output signals still have the right scaling as predicted by theory. On the other hand, when Euler discretization is used for $\mathbf{B}_l$, the latent states will have the correct scaling, but the output signals will explode. The results clearly highlight the importance of

correcting the scaling of the $\mu$P parameterization. When using the corrected scaling which we call $\mu$P-SSM, both the latent states and the output signals have the right scaling at initialization, and their updates have the same correct scaling. This holds true for both discretization schemes. Note that the slight shift in scaling when the width is small is due to finite sample effects. It stabilizes once the width is sufficiently large.

**Stability, generalization, and hyper-parameter transfer.** In Figure 3, we employ Mamba as a generative model on the wikitext-103 dataset and conduct single-epoch training for $20K$ iterations. We plot the test loss against the learning rate on a logarithmic scale and compare the results across different model widths (both $N_u$ and $N_x$). In this experiment, we use the standard setting where $N_u \gg N_x$ ($N_u = 16N_x$ in this case). Using $\mu$P-SSM scaling or $\mu$P (heuristic) significantly improves test performance compared to standard parameterization of Mamba for this task. For larger learning rate, $\mu$P-SSM shows better stability compared to $\mu$P (heuristic), highlighting the importance of deriving the correct scaling for SSMs rather than heuristically adopting $\mu$P (heuristic) without investigation. Furthermore, previously unreported, we observe stable HP transfer from small to large widths and monotonically improving performance with increasing model widths in structured SSMs. In contrast, we observe completely non-monotonic behavior under standard scaling of SSMs.

Note that optimal learning rate also appears to transfer under spectral scaling. The reasoning behind why optimal hyper-parameters transfer across scales is not completely understood. For instance, the optimal learning rate has been empirically shown to transfer across depth in transformers under appropriate depth-dependent scaling of the residual branches (Bordelon et al., 2023). However, from a theoretical standpoint, layers within each residual block of a transformer are in the lazy regime in the limit (Yang et al., 2023b). This is very similar to Mamba under spectral scaling. Under the ZOH discretization, the first block of the SSM is in the lazy regime but the outputs themselves are updated non-trivially. This suggests that a more thorough understanding of both necessary and sufficient conditions of the transferrability of hyper-parameters is warranted.

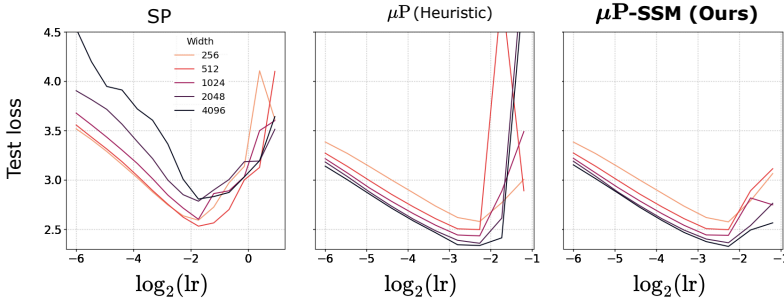

Figure 3: Test loss against learning rate on Mamba with varying widths ($N_u$ and $N_x$). Using $\mu$P-SSM scaling leads to substantially improved test performance compared to the SP scaling. Compared to $\mu$P (heuristic), $\mu$P-SSM scaling provides greater stability when utilizing large learning rates. Notably, we observe stable learning rate transfer from small to large model widths. Performance improves monotonically across widths in structured SSMs under $\mu$P-SSM scaling, as opposed to standard scaling where performance actually drops with scale after a certain width.

# 5  Related work

**Signal propagation.** Our work can be seen as scaling theory or signal propagation theory with the goal of preventing both vanishing and exploding signals in forward and backward passes. In this sense, we build on a rich literature, often restricted to an analysis at or close to initialization (Schoenholz et al., 2016, Poole et al., 2016, Hanin and Rolnick, 2018, Xiao et al., 2020). Towards understanding infinite-width limits of neural networks, kernel-based approaches (Neal, 1996, Jacot et al., 2018) and applications of mean-field theory (Mei et al., 2018) have yielded valuable insights.

**Tensor Programs.** Most promisingly, the Tensor Programs framework (Yang, 2019, Yang and Hu, 2021, Yang and Littwin, 2023, Yang et al., 2022, 2023b) covers many modern deep learning architectures, optimization algorithms and arbitrary $abc$-parameterizations. Each $abc$-parameterization is essentially defined by a layerwise scaling of initialization variance and learning rate as a function of

network width. Seminal work by Yang and Hu (2021) shows that there exists a unique maximal update parameterization ($\mu$P) that attains a stable feature learning infinite-width limit. This parameterization has since been shown to be a good model for understanding the properties of large models (Vyas et al., 2024), and has been extended to infinite width and depth limits of ResNets (Hayou et al., 2021, Li et al., 2021, Bordelon et al., 2023, Yang et al., 2023b) and Transformers (Noci et al., 2022, 2024).

**Structured SSMs.** Our analysis focuses on structured state space models (SSMs). The S4 model (Gu et al., 2021) is inspired by continuous-time linear SSMs, which are well-studied in control systems, and its specific initialization is motivated by the HiPPO theory (Gu et al., 2020). S4 and its variants e.g. DSS(Gu et al., 2022), S4D(Gupta et al., 2022), S5(Smith et al., 2022), etc., demonstrate impressive long-range dependency and overcome the quadratic computational cost of transformer models (Vaswani et al., 2017) w.r.t. sequence length. However, these models are less effective at modeling text, or even perform simple tasks such as selective copying (Gu and Dao, 2023). Mamba (Gu and Dao, 2023) is proposed to address such issues with selection mechanism. This line of work has also inspired revisiting Recurrent Neural Networks (RNNs) (Orvieto et al., 2023, De et al., 2024, Beck et al., 2024), leading to the growing interest in RNN-based sequence models.

# 6 Discussion

In this work, we study the scaling behavior of forward and backward signal propagation in structured state space models – a promising class of recent architectures. We show that existing scaling rules such as standard parameterization, $\mu P$, or spectral scaling conditions do not yield desirable properties such as feature learning in SSMs at scale. Through our analysis, we propose the correct scaling of state space models under which we empirically observe feature learning and transferability of hyper-parameters from small to large scale models.

**On Generalizing Tensor Programs.** While our proposed scaling has been derived by a thorough analysis of signal propagation in SSM layers, our results are still limited to the $N_u$ then $N_x$ tend to infinity setting. A completely rigorous analysis of SSMs in the proportional limit where $N_u$ and $N_x$ approach infinity with $\frac{N_x}{N_u}$ held roughly constant requires us to carefully track how the different activations and the updates are correlated with each other. For most standard architectures, the Tensor Program (TP) machinery provides the appropriate tools to do precisely this. Therefore, it is of considerable interest to generalize the TP framework. The key assumption that requires relaxation is that the different vectors in a TP (such as activations or updates) are asymptotically identically distributed. As discussed earlier, this assumption crucially underlies the key theoretical results of TP called Master Theorems. However, a potential path toward generalizing TP may be found by noting that the Master Theorems can be viewed as a non-linear compositional form of the law of large numbers or central limit theorem. Accordingly, it may be possible to relax the assumption of being identically distributed in the limit and instead ask that the entries of the vectors in a TP asymptotically satisfy weaker conditions such as Lindenberg or Kolmogorov conditions. Note however, that any such generalization is highly technical and is beyond the scope of the current work.

# References

Josh Achiam, Steven Adler, Sandhini Agarwal, Lama Ahmad, Ilge Akkaya, Florencia Leoni Aleman, Diogo Almeida, Janko Altenschmidt, Sam Altman, Shyamal Anadkat, et al. Gpt-4 technical report. *arXiv preprint arXiv:2303.08774*, 2023.

Maximilian Beck, Korbinian Pöppel, Markus Spanring, Andreas Auer, Oleksandra Prudnikova, Michael Kopp, Günter Klambauer, Johannes Brandstetter, and Sepp Hochreiter. xlstm: Extended long short-term memory. *arXiv preprint arXiv:2405.04517*, 2024.

Blake Bordelon, Lorenzo Noci, Mufan Bill Li, Boris Hanin, and Cengiz Pehlevan. Depthwise hyperparameter transfer in residual networks: Dynamics and scaling limit. *arXiv:2309.16620*, 2023.

Tom Brown, Benjamin Mann, Nick Ryder, Melanie Subbiah, Jared D Kaplan, Prafulla Dhariwal, Arvind Neelakantan, Pranav Shyam, Girish Sastry, Amanda Askell, et al. Language models are few-shot learners. *Advances in neural information processing systems*, 33:1877–1901, 2020.

Aakanksha Chowdhery, Sharan Narang, Jacob Devlin, Maarten Bosma, Gaurav Mishra, Adam Roberts, Paul Barham, Hyung Won Chung, Charles Sutton, Sebastian Gehrmann, et al. Palm: Scaling language modeling with pathways. *Journal of Machine Learning Research*, 24(240):1–113, 2023.

Soham De, Samuel L Smith, Anushan Fernando, Aleksandar Botev, George Cristian-Muraru, Albert Gu, Ruba Haroun, Leonard Berrada, Yutian Chen, Srivatsan Srinivasan, et al. Griffin: Mixing gated linear recurrences with local attention for efficient language models. *arXiv preprint arXiv:2402.19427*, 2024.

Jacob Devlin, Ming-Wei Chang, Kenton Lee, and Kristina Toutanova. Bert: Pre-training of deep bidirectional transformers for language understanding. *arXiv preprint arXiv:1810.04805*, 2018.

Gemini Team, Rohan Anil, Sebastian Borgeaud, Yonghui Wu, Jean-Baptiste Alayrac, Jiahui Yu, Radu Soricut, Johan Schalkwyk, Andrew M Dai, Anja Hauth, et al. Gemini: a family of highly capable multimodal models. *arXiv preprint arXiv:2312.11805*, 2023.

Albert Gu and Tri Dao. Mamba: Linear-time sequence modeling with selective state spaces. *arXiv preprint arXiv:2312.00752*, 2023.

Albert Gu, Tri Dao, Stefano Ermon, Atri Rudra, and Christopher Ré. Hippo: Recurrent memory with optimal polynomial projections. In *Advances in Neural Information Processing Systems*, volume 33, pages 1474–1487, 2020.

Albert Gu, Karan Goel, and Christopher Re. Efficiently modeling long sequences with structured state spaces. In *International Conference on Learning Representations*, 2021.

Albert Gu, Karan Goel, Ankit Gupta, and Christopher Ré. On the parameterization and initialization of diagonal state space models. *Advances in Neural Information Processing Systems*, 35:35971–35983, 2022.

Ankit Gupta, Albert Gu, and Jonathan Berant. Diagonal state spaces are as effective as structured state spaces. *Advances in Neural Information Processing Systems*, 35:22982–22994, 2022.

Boris Hanin and David Rolnick. How to start training: The effect of initialization and architecture. *Advances in neural information processing systems*, 31, 2018.

Soufiane Hayou, Eugenio Clerico, Bobby He, George Deligiannidis, Arnaud Doucet, and Judith Rousseau. Stable resnet. In *International Conference on Artificial Intelligence and Statistics*, pages 1324–1332. PMLR, 2021.

Arthur Jacot, Franck Gabriel, and Clément Hongler. Neural Tangent Kernel: Convergence and generalization in neural networks. In *Advances in Neural Information Processing Systems (NeurIPS)*, pages 8571–8580, 2018.

Mufan Li, Mihai Nica, and Dan Roy. The future is log-gaussian: Resnets and their infinite-depth-and-width limit at initialization. In *Advances in Neural Information Processing Systems (NeurIPS)*, volume 34, pages 7852–7864, 2021.

Song Mei, Andrea Montanari, and Phan-Minh Nguyen. A mean field view of the landscape of two-layer neural networks. *Proceedings of the National Academy of Sciences*, 115(33):E7665–E7671, 2018.

Stephen Merity, Caiming Xiong, James Bradbury, and Richard Socher. Pointer sentinel mixture models. In *International Conference on Learning Representations*, 2016.

Radford M. Neal. *Priors for Infinite Networks*, pages 29–53. Springer New York, 1996.

Lorenzo Noci, Sotiris Anagnostidis, Luca Biggio, Antonio Orvieto, Sidak Pal Singh, and Aurelien Lucchi. Signal propagation in transformers: Theoretical perspectives and the role of rank collapse. *Advances in Neural Information Processing Systems*, 35:27198–27211, 2022.

Lorenzo Noci, Chuning Li, Mufan Li, Bobby He, Thomas Hofmann, Chris J Maddison, and Dan Roy. The shaped transformer: Attention models in the infinite depth-and-width limit. *Advances in Neural Information Processing Systems*, 36, 2024.

Antonio Orvieto, Samuel L Smith, Albert Gu, Anushan Fernando, Caglar Gulcehre, Razvan Pascanu, and Soham De. Resurrecting recurrent neural networks for long sequences. In *International Conference on Machine Learning*, pages 26670–26698. PMLR, 2023.

Ben Poole, Subhaneil Lahiri, Maithra Raghu, Jascha Sohl-Dickstein, and Surya Ganguli. Exponential expressivity in deep neural networks through transient chaos. *Advances in neural information processing systems*, 29, 2016.

Alec Radford, Jeff Wu, Rewon Child, David Luan, Dario Amodei, and Ilya Sutskever. Language models are unsupervised multitask learners. 2019. URL https://api.semanticscholar.org/CorpusID:160025533.

Samuel S Schoenholz, Justin Gilmer, Surya Ganguli, and Jascha Sohl-Dickstein. Deep information propagation. *arXiv:1611.01232*, 2016.

Pranab K Sen and Julio M Singer. *Large sample methods in statistics: an introduction with applications*, volume 25. CRC press, 1994.

Jimmy TH Smith, Andrew Warrington, and Scott Linderman. Simplified state space layers for sequence modeling. In *The Eleventh International Conference on Learning Representations*, 2022.

Hugo Touvron, Louis Martin, Kevin Stone, Peter Albert, Amjad Almahairi, Yasmine Babaei, Nikolay Bashlykov, Soumya Batra, Prajjwal Bhargava, Shruti Bhosale, et al. Llama 2: Open foundation and fine-tuned chat models. *arXiv preprint arXiv:2307.09288*, 2023.

Ashish Vaswani, Noam Shazeer, Niki Parmar, Jakob Uszkoreit, Llion Jones, Aidan N Gomez, Łukasz Kaiser, and Illia Polosukhin. Attention is all you need. *Advances in neural information processing systems*, 30, 2017.

Roman Vershynin. Spectral norm of products of random and deterministic matrices. *Probability theory and related fields*, 150(3):471–509, 2011.

Nikhil Vyas, Alexander Atanasov, Blake Bordelon, Depen Morwani, Sabarish Sainathan, and Cengiz Pehlevan. Feature-learning networks are consistent across widths at realistic scales. *Advances in Neural Information Processing Systems*, 36, 2024.

Thomas Wolf, Lysandre Debut, Victor Sanh, Julien Chaumond, Clement Delangue, Anthony Moi, Pierric Cistac, Tim Rault, Rémi Louf, Morgan Funtowicz, et al. Huggingface's transformers: State-of-the-art natural language processing. *arXiv preprint arXiv:1910.03771*, 2019.

Lechao Xiao, Jeffrey Pennington, and Samuel Schoenholz. Disentangling trainability and generalization in deep neural networks. In *International Conference on Machine Learning*, pages 10462–10472. PMLR, 2020.

Greg Yang. Wide feedforward or recurrent neural networks of any architecture are gaussian processes. *Advances in Neural Information Processing Systems*, 32, 2019.

Greg Yang and Edward J. Hu. Tensor programs iv: Feature learning in infinite-width neural networks. In *International Conference on Machine Learning (ICML)*, 2021.

Greg Yang and Etai Littwin. Tensor programs ivb: Adaptive optimization in the infinite-width limit. *arXiv:2308.01814*, 2023.

Greg Yang, Edward J Hu, Igor Babuschkin, Szymon Sidor, Xiaodong Liu, David Farhi, Nick Ryder, Jakub Pachocki, Weizhu Chen, and Jianfeng Gao. Tensor programs v: Tuning large neural networks via zero-shot hyperparameter transfer. *arXiv:2203.03466*, 2022.

Greg Yang, James B Simon, and Jeremy Bernstein. A spectral condition for feature learning. *arXiv preprint arXiv:2310.17813*, 2023a.

Greg Yang, Dingli Yu, Chen Zhu, and Soufiane Hayou. Tensor programs vi: Feature learning in infinite-depth neural networks. *arXiv preprint arXiv:2310.02244*, 2023b.

# A Background

## A.1 NE⊗OR⊤, abc-parameterizations and μP

**NE⊗OR⊤ programs.** For the precise definitions and theorems of the NE⊗OR⊤ framework we refer to Yang and Littwin (2023). Here we try to provide an intuitive introduction.

A NE⊗OR⊤ program consists of a set of vectors inductively generated from a set of initial matrices, vectors, and scalars following a set of allowed instructions: vector averages (Avg), matrix multiplications (MatMul), and non-linear outer products (OuterNonlin).

All entries of an initial vector $v$ are sampled iid from $N(0, 1)$. Every entry of an initial matrix $A$ is sampled independently from distributions that all have mean $0$, variance $n^{-1}$ and that satisfy a technical boundedness assumption on all higher-order moments. Nonlinearities $\psi$ used for OuterNonlin operations are either pseudo-Lipschitz or polynomially smooth. Initial scalars $c$ converge to $0$ a.s.

New scalars $c$ can be introduced into the program by the Avg operation over a vector $v$, defined as $c = \frac{1}{n}\sum_\alpha^n v_\alpha$. Now the NE⊗OR⊤ Master Theorem implies $c \to \mathring{c} := \mathbb{E}Z^v$ almost surely. Matrix multiplications $Av$ are defined as usual. In the limit, a non-trivial interaction term arises if $A^\top$ had appeared in a NE⊗OR⊤ operation that has generated the vector $v$. To define the OuterNonlin operation, aggregate a fixed amount $|\mathbf{x}|$ of NE⊗OR⊤ vectors in $\mathbf{x}$ and fix $r \in \mathbb{N}$. Denote the collection of all $l$ defined NE⊗OR⊤ scalars by $\mathbf{c}$. Then a OuterNonlin operation with nonlinearity $\psi : \mathbb{R}^{|\mathbf{x}|(r+1)+l} \to \mathbb{R}$, is given by

$$y_\alpha = \frac{1}{n^r} \sum_{\beta_1,\ldots,\beta_r}^n \psi(\mathbf{x}_\alpha, \mathbf{x}_{\beta_1}, \ldots, \mathbf{x}_{\beta_r}; \mathbf{c}),$$

and, due to the NE⊗OR⊤ Master Theorem, behaves in the limit as

$$Z^{\mathbf{y}} = f(Z^{\mathbf{x}}), \quad \text{where } f : \mathbb{R}^{|\mathbf{x}|} \to \mathbb{R}, \quad f(Z^{\mathbf{x}}) = \mathbb{E}[\psi(Z^{\mathbf{x}}, Z_1^{\mathbf{x}}, \ldots, Z_r^{\mathbf{x}}, \mathring{c})],$$

where $Z^{\mathbf{x}}$ is an $r$-dimensional random vector with the limiting distribution of $\mathbf{x}$, and $Z_1^{\mathbf{x}}, \ldots, Z_r^{\mathbf{x}}$ are iid copies of $Z^{\mathbf{x}}$.

**The Maximal Update Parameterization (μP).** For any architecture/component whose forward pass is representable as a NE⊗OR⊤ program, μP can be derived following the instructions below. Let each parameter tensor $W$ be parameterized as $W = n^{-a_W} w$ where $w$ is a trainable parameter with initialization $w_{\alpha\beta} \sim \mathcal{N}(0, n^{-2b_W})$. Let the learning rate be parameterized as $\eta n^{-c}$ for some width-independent $\eta > 0$. Then μP is prescribed as follows: Set $c = 0$ and $b_W = 1/2$ for all parameter tensors $W$.

- $a_W = 0$ if the input and output to $W$ are width-independent (e.g., scalars),
- $a_W = 2$ if both input and output to $W$ scale with width (e.g., attention matrices).
- $a_W = 1$ if input is width-independent and output scales with width (e.g., input embeddings).
- $a_W = 1/2$ if input scales with width and output is width-independent (e.g., output matrices).

## A.2 Classical Limit Theorems

Here we recapitulate classical results about large sums of random variables that we use in our proofs. We present the versions of the Kolmogorov strong law of large numbers and the Lindenberg-Feller Central Limit Theorem provided in Sen and Singer (1994, Theorems 3.2.10 and 3.3.3).

**Theorem A.1** (Kolmogorov Strong Law of Large Numbers). *Let $X_i$, $i \geq 1$, be independent random variables such that $\mathbb{E}X_i = \mu_i$ and $Var(X_i) = \sigma_i^2$ exist for every $i \geq 1$. Then*

$$\sum_{k \geq 1} k^{-2}\sigma_k^2 < \infty \quad \implies \quad n^{-1}\sum_{i=1}^n X_i - n^{-1}\sum_{i=1}^n \mu_i \xrightarrow{a.s.} 0.$$

**Theorem A.2** (Lindenberg-Feller-Central Limit Theorem). *Let $X_i$, $i \geq 1$, be independent random variables such that $\mathbb{E}X_i = \mu_i$ and $Var(X_i) = \sigma_i^2$ exist for every $i \geq 1$. Also let $s_n^2 = \sum_{i=1}^n \sigma_i^2$ and $Z_n = s_n^{-1}(\sum_{i=1}^n X_i - \sum_{i=1}^n \mu_i)$. Then the Lindenberg-Feller condition*

$$\forall \varepsilon > 0, \qquad \frac{1}{s_n^2}\sum_{i=1}^n \mathbb{E}\left[(X_i - \mu_i)^2 \mathbb{1}_{\{|X_i - \mu_i| \geq \varepsilon s_n\}}\right] \to 0, \quad as \quad n \to \infty,$$

*holds if and only if both*

*(i)* $\max_{1 \le i \le n} \frac{\sigma_i^2}{s_n^2} \to 0$ *as* $n \to \infty$, *and*

*(ii)* $Z_n$ *converges in distribution to a Gaussian with mean* $0$ *and standard deviation* $1$.

## B  Definitions

**Definition B.1** (Softplus). The *softplus function* is defined as $\mathrm{Softplus}(x) = \frac{1}{\beta} \log(1 + \exp(\beta x))$ with default smoothing value $\beta = 1$.

### B.1  Feature Learning in General Sequence Models

Let $\mathcal{G}$ be a directed acyclic graph (DAG) with vertices $\mathcal{V}(\mathcal{G})$ and edges $\mathcal{E}(\mathcal{G})$. For each node $\mathbf{v} \in \mathcal{V}(\mathcal{G})$, its parent nodes are denoted by $\mathcal{PA}(\mathbf{v})$. In a sequence model, each node $\mathbf{v}$ corresponds to a sequence $\mathbf{v}_{1:L}$ with length $L$, computed from its parent node sequences with operation $\mathbf{v}_{1:L} = f_v(\{\mathbf{u}_{1:L} | \mathbf{u} \in \mathcal{PA}(\mathbf{v})\})$. These sequence layers can be shared instance-wise multi-layer perceptrons (MLPs), normalizations, residual summation, or transformers/recurrent layers, etc.

**Definition B.2** (Feature Learning in Sequence Models). A sequence model is in the feature learning regime if for any $\mathbf{v} \in \mathcal{V}(\mathcal{G})$, the features $\mathbf{v}_{1:L}$ where $\mathbf{v} \in \mathbb{R}^{N_v}$ and the updates of features $\Delta \mathbf{v}_{1:L}$ after one gradient update have the following scaling:

$$\exists\, l \in [L],\ \|\mathbf{v}_l\|_2 = \Theta(\sqrt{N_v}) \quad \text{(Stability at initialization)} \tag{9}$$

$$\exists\, l \in [L],\ \|\Delta \mathbf{v}_l\|_2 = \Theta(\sqrt{N_v}) \quad \text{(Non-trivial feature updates)} \tag{10}$$

## C  Proofs

### C.1  Structured SSMs are not covered by previous approaches

**Proposition C.1** (**Structured SSMs are not generally representable as** NE⊗OR⊤ **programs**). *The forward and backward signal propagation of structured SSMs including S4 and MAMBA trained via standard algorithms such as SGD or ADAM are not representable as a* NE⊗OR⊤ *program. Indeed this holds for all existing Hippo-parameterizations of the structured matrix A including HiPPO-LegS, HiPPO-LegS-N, HiPPO-LegS-D, S4D-Inv, S4D-Lin, and S4D-Real.*

*Proof.* Recall the definition of a NE⊗OR⊤ program (Yang and Littwin, 2023). Let $\mathbf{A}$ be any of the HiPPO or S4D structured matrices. As the entries of $\mathbf{A}$ at initialization are deterministic and differ, $\mathbf{A}$ can neither be an initial NE⊗OR⊤ matrix, nor be generated by an OuterNonlin of random NE⊗OR⊤ vectors (acting on all coordinate dimensions in the same way). This is because no allowed NE⊗OR⊤ operation can generate a NE⊗OR⊤ vector with differing variance in its coordinates, whereas the multiplication with $\mathbf{A}$ clearly produces differing variances in each entry. All that is left to do is to show this claim via induction.

Claim: All NE⊗OR⊤ vectors have the same variance in each coordinate.

To start the induction, all entries of an initial vector $v$ are sampled iid from $N(0, 1)$.

Now assume all vectors $v$ currently defined in the NE⊗OR⊤ program have the same variance. The allowed operations to generate a NE⊗OR⊤ vector are matrix multiplication with an initial matrix and OuterNonlin. As initial matrices $\mathbf{A}_0$ have independent entries with mean $0$ and variance $n^{-1}$, the multiplication $\mathbf{A}_0 v$ will again generate a vector with the same variance in each coordinate.

An OuterNonlin operation takes in previously defined NE⊗OR⊤ vectors and treats all coordinates in the same way. Consequently it will again generate a NE⊗OR⊤ vector with the same variance in each coordinate.

There is no other way to generate a new NE⊗OR⊤ vector, which concludes the induction.

To see that the structured matrix $\mathbf{A}$ produces nonisotropic coordinate distributions, we detail the argument for an S4 recurrent layer. The other choices of $\mathbf{A}$ follow via analogous arguments. Recall

that the S4 recurrent layer $\mathbf{y}_{1:L} = f_{S4}(\mathbf{u}_{1:L}; \mathbf{w} = \{\mathbf{B}, \mathbf{C}\})$ can be viewed as a discretization of a continuous-time SSM given by

$$\frac{d}{dt}\mathbf{x}_t = \mathbf{A}\mathbf{x}_t + \mathbf{B}\mathbf{u}_t \quad \text{and} \quad \mathbf{y}_t = \text{Re}[\mathbf{C}\mathbf{x}_t] \tag{11}$$

for $l = 1, \ldots, L$ with $\mathbf{x}_0 = 0$ and where $\text{Re}(\cdot)$ gives the real part of a complex vector, and where we let $\mathbf{A} = \text{diag}(\mathbf{a})$ with $\mathbf{a} \in \mathbb{C}^{N_x}$, $\mathbf{B} \in \mathbb{C}^{N_x \times N_u}$, $\mathbf{C} \in \mathbb{C}^{N_y \times N_x}$. We have the discretized sequence to sequence mapping $f_{S4}$ as follows:

$$\mathbf{x}_l = \mathbf{A}'\mathbf{x}_{l-1} + \mathbf{B}'\mathbf{u}_l \quad \text{and} \quad \mathbf{y}_l = \text{Re}[\mathbf{C}'\mathbf{x}_l] \tag{12}$$

where we follow the Zero-Order-Hold (ZOH) discretization method,

$$\mathbf{A}' = \exp(\tau \cdot \mathbf{A}), \quad \mathbf{B}' = (\mathbf{A}' - \mathbf{I})\mathbf{A}^{-1}\mathbf{B}, \quad \text{and} \quad \mathbf{C}' = \mathbf{C}. \tag{13}$$

To model long-range dependency, S4 advocates for HiPPO theory (Gu et al., 2020). While there are many possible Hippo-based initializations possible for the structured transition matrices (e.g., Hippo-Leg-S, S4D-Inv, and S4D-Real) and our result holds for all the parameterizations, let us consider a simplified initialization for clarity. Following S4D-Lin, we can set $a_n = -\frac{1}{2} + i\pi n$. $\mathbf{B}$ and $\mathbf{C}$ are initialized such that the entries are i.i.d and follow a Gaussian distribution with mean 0 and variance $\sigma_B^2$ and $\sigma_C^2$ respectively (for both real and imaginary parts).

Assume for now that the inputs to the SSM layer are i.i.d which holds asymptotically in general say if they are outputs of a previous layer such as an MLP. Let us consider the distribution of the hidden states for token index 0 at initialization: $\mathbf{x}_0 = \mathbf{B}'\mathbf{u}_0$, where $\mathbf{B}' = (\mathbf{A}' - \mathbf{I})\mathbf{A}^{-1}\mathbf{B}$, and $\mathbf{A}' = \text{diag}(a'_1, \ldots, a'_{N_x})$ with $a'_n = e^{-\frac{1}{2}\tau}(\cos(\tau\pi n) + i\sin(\tau\pi n))$, then the eigenvalues of $\mathbf{A}'$ are clearly $\Theta(1)$. To further simplify, if we set $\tau = 2$, then $\|\mathbf{A}'\|_* = e^{-1}$ and for all complex vectors $\mathbf{v} \in \mathbb{C}^{N_x}$ it holds that $\mathbf{A}'\mathbf{v} = -e^{-1}\mathbf{v}$. Since $B$ is an i.i.d Gaussian matrix, for sufficiently large width, $\mathbf{B}\mathbf{u}_0$ is distributed i.i.d by a simple Central Limit Theorem argument. However, $\mathbf{B}'$ also depends on $\mathbf{A}^{-1} = \text{diag}(a_n^{-1})$ with $a_n^{-1} = -\frac{1+2\pi ni}{1/2+2\pi^2 n^2}$. Due to the linear decay of the diagonal entries of $\mathbf{A}^{-1}$, the hidden states $x_0$ already at initialization are not (even asymptotically) identically distributed. Therefore, there cannot exist a coordinate distribution that represents the entries of vectors arising in modern SSMs such as MAMBA neither at initialization nor over the course of training.

$\square$

**Proposition C.2 (Spectral scaling does not generally imply feature learning in SSMs).** *Structured SSMs including S4 and MAMBA trained via standard algorithms such as SGD or ADAM that satisfy spectral scaling conditions $(*)$ do not satisfy conditions for feature learning given in Equation $(\Delta)$. Indeed this holds for all well-known Hippo-parameterizations of the structured matrix $\mathbf{A}$ including HiPPO-LegS, HiPPO-LegS-N, HiPPO-LegS-D, S4D-Inv, S4D-Lin, and S4D-Real.*

*Proof.* Let $\mathbf{A}$ be any of the HiPPO or S4D structured matrices.

Consider the scale of the hidden states at initialization for token index $0$. By definition, $\|\mathbf{x}_1\|_2 = \|\mathbf{B}'\mathbf{u}_1\|_2 = \|\mathbf{\Lambda}\mathbf{B}\mathbf{u}_1\|_2$, where $\mathbf{\Lambda} = (\mathbf{A}' - \mathbf{I})\mathbf{A}^{-1}$. Then, observe that $\|\mathbf{B}'\mathbf{u}_1\|_2^2 = \sum_{i=1}^{N_x} \Lambda_i^2 \left(\sum_{j=1}^{N_u} B_{i,j}u_{1,j}\right)^2$ is a sum of $N_x$ independent (but not identically distributed) random variables. However, it is easy to verify that they satisfy the Kolmogorov condition and therefore the sum behaves according to the strong law of large numbers. Intuitively, this is allowed by the polynomial decay of eigenvalues of $\mathbf{\Lambda}$ and the correct scaling of the inputs to the S4 layer. Applying the Kolmogorov SLLN (Theorem A.1), we obtain

$$\sum_{i=1}^{N_x} \Lambda_i^2 \left(\sum_{j=1}^{N_u} B_{i,j}u_{1,j}\right)^2 \xrightarrow{a.s.} \sigma_B^2 \|\mathbf{u}_1\|^2 c\zeta(2),$$

where $\zeta(2)$ denotes the Riemann zeta function at 2 and equates to $\pi^2/6$, and $c > 0$ is some width-independent constant. The zeta function appears due to $c\zeta(2) \leq \sum_{i=1}^{N_x} \Lambda_i^2 \leq \sum_{i=1}^{N_x} \frac{1}{(i+1)^2} \leq \zeta(2)$ for $N_x$ large enough (see (37) for more details). Here and throughout the paper, for the formal application of the LLN or the CLT, we first adequately normalize all quantities to arrive at a well-defined width-independent limit statements, but do not write out such technicalities for conciseness.

Observe that, for spectral scaling conditions to yield the right scaling of the initialization variance, it is crucial that the following condition holds:

$$\|\mathbf{B}'\mathbf{u}_1\|_2 \in \Theta\left(\|\mathbf{B}'\|_* \|\mathbf{u}_1\|_2\right). \tag{14}$$

Since the spectrum of $(\mathbf{A}' - \mathbf{I})\mathbf{A}^{-1}$ is less than 1, an upper bound on the spectral norm of $\mathbf{B}' = (\mathbf{A}' - \mathbf{I})\mathbf{A}^{-1}\mathbf{B}$ can be found in Vershynin (2011) and is given by $c(\sqrt{N_x} + \sqrt{N_u})$ for some width-independent constant $c$. A matching lower bound can be easily found by noting that the spectral norm is lower bounded by the maximal row and column norm: $\|\mathbf{B}'\|_* \geq \max\left\{\max_i \|\mathbf{B}'_{i:}\|_2, \max_j \|\mathbf{B}'_{:j}\|_2\right\} \geq \sqrt{N_u}\sigma_B$. Therefore $\frac{\|\mathbf{B}'\mathbf{u}\|_2}{(\|\mathbf{B}'\|_* \|\mathbf{u}_1\|_2)} \in \Theta(\frac{1}{\sqrt{N_u}})$ which clearly violates (14). This spectral decay is induced by all of the considered choices of $A$.

$\square$

## C.2   S6 Mamba recurrent layer

Recall that a S6 Mamba layer $\mathbf{y}_{1:L} = f_{\text{Mamba}}(\mathbf{u}_{1:L})$ can be written as follows:

$$\mathbf{B}_l = \text{Lin}_{N_x}(\mathbf{u}_l; \mathbf{W}_B, \mathbf{b}_B) \in \mathbb{R}^{N_x}, \tag{15}$$

$$\mathbf{C}_l = \text{Lin}_{N_y}(\mathbf{u}_l; \mathbf{W}_C, \mathbf{b}_C) \in \mathbb{R}^{N_x}, \tag{16}$$

$$\tau_l = \text{Softplus}(\tau_0 + \text{Broadcast}_{N_u}(\text{Lin}_1(\mathbf{u}_l; \mathbf{W}_\tau, \mathbf{b}_\tau))), \ \tau_0, \tau_l \in \mathbb{R}^{N_u}. \tag{17}$$

For $i = 1, \ldots, N_u$, we have $N_u$ 1D SSMs:

$$\mathbf{A}^{(i)} = \text{Diag}(-\exp(\mathbf{a}_{\log}^{(i)})), \ \mathbf{a}_{\log}^{(i)} \in \mathbb{R}^{N_x}, \quad \mathbf{x}_l^{(i)} = \mathbf{A}_l^{'(i)}\mathbf{x}_{l-1}^{(i)} + u_l^{(i)}\mathbf{B}_l^{'(i)}, \ u_l^{(i)} \in \mathbb{R}, \ \mathbf{x}_l^{(i)} \in \mathbb{R}^{N_x},$$

$$\mathbf{A}_l^{'(i)}, \mathbf{B}_l^{'(i)} = \text{ZOH}(\tau_l^{(i)}, \mathbf{A}^{(i)}, \mathbf{B}_l), \ \tau_l^{(i)} \in \mathbb{R}, \qquad y_l^{(i)} = \mathbf{C}_l^{\mathsf{T}}\mathbf{x}_l^{(i)}, \ y_l^{(i)} \in \mathbb{R}.$$

where $\mathbf{x}_0^{(i)} = \mathbf{0}$, $\mathbf{a}_{\log}^{(i)}$, $\tau_0 \in \mathbb{R}^{N_u}$, and all linear layer weights $\mathbf{W}_B, \mathbf{b}_B, \mathbf{W}_C, \mathbf{b}_C, \mathbf{W}_\tau, \mathbf{b}_\tau$ are trainable parameters. The Zero-Order-Hold (ZOH) discretization procedure $\mathbf{A}_l^{'(i)}, \mathbf{B}_l^{'(i)} = \text{ZOH}(\tau, \mathbf{A}^{(i)}, \mathbf{B}_l^{'(i)})$ can be written as:

$$\mathbf{A}_l^{'(i)} = \exp(\tau_l^{(i)} \cdot \mathbf{A}^{(i)}), \qquad \mathbf{B}_l^{'(i)} = (\mathbf{A}_l^{'(i)} - \mathbf{I})\mathbf{A}^{(i)^{-1}}\mathbf{B}_l^{(i)}. \tag{18}$$

In a nutshell, Mamba can be seen as $N_u$ SSMs, one for each input channel. Weights for these SSMs are shared and depend on the input at that recurrent step. Because of the dependency on inputs, Mamba can model non-stationary sequences. Weight matrices are initialized as follows:

$$\mathbf{W}_B \sim \mathcal{N}(\mathbf{0}, \sigma_B^2\mathbf{I}) \in \mathbb{R}^{N_x \times N_u}, \ \mathbf{W}_C \sim \mathcal{N}(\mathbf{0}, \sigma_C^2\mathbf{I}) \in \mathbb{R}^{N_x \times N_u}, \ \mathbf{W}_\tau \sim \mathcal{N}(\mathbf{0}, \sigma_\tau^2\mathbf{I}) \in \mathbb{R}^{1 \times N_u},$$

and zero initialization for all biases. For each SSM, $\mathbf{A}^{(i)}$ is still initialized according to the HiPPO theory. When $\mathbf{A}^{(i)}$ is real-valued, we let $\mathbf{a}_{\log}^{(i)}[j] = \log(j+1)$ and $\tau_0$ can be seen a bias term initialized to $\tau_0 \sim \text{Softplus}^{-1}(\mathcal{U}(0.001, 0.1))$.

Below we restate Propositions 3.2 and 3.3 followed by their proofs.

**Proposition 3.2** (**Scale of hidden states** $\mathbf{x}_1^{(i)}$ **in Mamba at initialization**). *Under the ZOH discretization procedure, as $N_u$ then $N_x$ approach infinity, for any $i \in [N_u]$, the squared $l_2$-norm of the hidden states $\|\mathbf{x}_1^{(i)}\|_2^2$ is a.s. scaled as $\|\mathbf{x}_1^{(i)}\|_2^2 \in \Theta\left(\zeta(2)\,\sigma_B^2\,\|\mathbf{u}_1\|^2\,(u_1^{(i)})^2\right)$, where $\zeta(2)$ denotes the Riemann zeta function at 2.*

**Proposition 3.3** (**Scale of outputs** $\mathbf{y}_1^{(i)}$ **of a Mamba layer at initialization**). *Under the ZOH discretization procedure as $N_u$ then $N_x$ approach infinity, for any $i \in [N_u]$, the output $y_1^{(i)}$ converges in distribution to a Gaussian with mean 0 and standard deviation $C\sigma_B\sigma_C\|\mathbf{u}_1\|_2^2$ for some width-independent constant $C > 0$.*

***Proofs for Proposition 3.2 and Proposition 3.3.*** For the following 1D linear state space model:

$$\begin{cases} \mathbf{x}_l^{(i)} = \mathbf{A}_l^{'(i)}\mathbf{x}_{l-1}^{(i)} + \mathbf{B}_l^{'(i)}u_l^{(i)}, & u_l^{(i)} \in \mathbb{R}, \ \mathbf{x}_l^{(i)} \in \mathbb{R}^{N_x}, \\ y_l^{(i)} = \mathbf{C}_l^{\mathsf{T}}\mathbf{x}_l^{(i)}, \ y_l^{(i)} \in \mathbb{R}. \end{cases} \tag{19}$$

Following condition (INIT), the 1D SSM admits stability at initialization, when the following condition is satisfied:

$$\text{If } |u_l^{(i)}| \in \Theta(1) \text{ then } \|\mathbf{x}_l^{(i)}\| \in \Theta(\sqrt{N_x}) \text{ and } |y_l^{(i)}| \in \Theta(1).$$

For simplicity of exposition, let's begin by considering the scale of $\mathbf{x}_1^{(i)} = (\mathbf{A}_l^{'(i)} - \mathbf{I})(\mathbf{A}^{(i)})^{-1}\mathbf{B}_1^{(i)}u_l^{(i)}$ before generalizing to arbitrary $l \in [L]$.

Since $\mathbf{B_1} = \mathbf{W}_B\mathbf{u}_1 + \mathbf{b}_B$ with $\mathbf{b}_B = \mathbf{0}$ at initialization,

$$\left\|\mathbf{x}_1^{(i)}\right\|_2^2 = \left\|(\mathbf{A}_l^{'(i)} - \mathbf{I})(\mathbf{A}^{(i)})^{-1}\mathbf{W}_B\mathbf{u}_1 u_1^{(i)}\right\|_2^2 = \sum_{m=1}^{N_x}(\Lambda_i)_{m,m}^2 (u_1^{(i)})^2 \big(\sum_{n=1}^{N_u}(\mathbf{W}_B)_{m,n}(u_1^{(n)})\big)^2,$$

where $\Lambda_i = (\mathbf{A}_l^{'(i)} - \mathbf{I})(\mathbf{A}^{(i)})^{-1}$.

Note that the inner summation can be expressed as

$$\big(\sum_{n=1}^{N_u}(\mathbf{W}_B)_{m,n}(u_1^{(n)})\big)^2 = \sum_{n=1}^{N_u}(\mathbf{W}_B)_{m,n}^2(u_1^{(n)})^2 + \sum_{n'\neq n''=1}^{N_u}(\mathbf{W}_B)_{m,n'}u_1^{(n')}(\mathbf{W}_B)_{m,n''}u_1^{(n'')}.$$

Since $\mathbf{W}_B$ has i.i.d Gaussian entries and due to Assumption 3.1, as $N_u \to \infty$, the first term behaves according to law of large numbers and converges almost surely to $\sigma_B^2\|\mathbf{u}_1\|^2$. The second term behaves according to central limit theorem and converges in distribution to a Gaussian with mean 0 and variance $\sigma_B^4 \sum_{n'\neq n''=1}^{N_u} u_1^{(n')}u_1^{(n'')}$. Therefore, as $N_u \to \infty$, $\big(\sum_{n=1}^{N_u}(\mathbf{W}_B)_{m,n}(u_1^{(n)})\big)^2$ converges to a Gaussian distribution with mean $\sigma_B^2\|\mathbf{u}_1\|^2$ and variance $\sigma_B^4 \sum_{n'\neq n''=1}^{N_u} u_1^{(n')}u_1^{(n'')}$.

For applying **Kolmogorov SLLN** (Theorem A.1), note that $c \cdot \zeta(2) \leq \sum_{m=1}^{N_x}(\Lambda_i)_{m,m}^2 \leq \sum_{m=1}^{N_x}\frac{1}{(i+1)^2} \leq \zeta(2)$ and $c \cdot \zeta(4) \leq \sum_{m=1}^{N_x}(\Lambda_i)_{m,m}^4 \leq \sum_{m=1}^{N_x}\frac{1}{(i+1)^4} \leq \zeta(4)$ for some constant $c \in (0,1)$ and for $N_x$ large enough (see (37) for more details). For random variables $v_m \sim \mathcal{N}(\sigma_B^2, C^2)$ for some $C^2 > 0$, it holds that $Var((\Lambda_i)_{m,m}^2 v_m) \leq \mathbb{E}[(\Lambda_i)_{m,m}^4 v_m^2] \leq (\Lambda_i)_{m,m}^4(\sigma_B^4 + C^2)$, so that the Kolmogorov condition is fulfilled, and we have that as $N_u$ then $N_x$ approach infinity, for some width-independent constant $c > 0$,

$$\sum_{m=1}^{N_x}(\Lambda_i)_{m,m}^2(u_1^{(i)})^2\big(\sum_{n=1}^{N_u}(\mathbf{W}_B)_{m,n}(u_1^{(n)})\big)^2 \xrightarrow{a.s.} \sigma_B^2(u_1^{(i)})^2\|\mathbf{u}_1\|^2 c\,\zeta(2).$$

Therefore, for stability at initialization, the scale of initialization $\sigma_B \in \Theta(\sqrt{\frac{N_x}{N_u}})$.

**Scale of $y_l^{(i)}$.** When $l = 1$, $y_1^{(i)} = \mathbf{C}_1\mathbf{x}_1^{(i)}$, where $\mathbf{x}_1^{(i)} = \mathbf{B}_1^{'(i)}u_1^{(i)}$ and we have

$$y_1^{(i)} = u_1^{(i)}\sum_{m=1}^{N_x}(\Lambda_i)_{m,m}\sum_{j,k=1}^{N_u}(\mathbf{W}_C)_{m,j}(\mathbf{W}_B)_{m,k}u_1^{(j)}u_1^{(k)}$$

Applying the **Lindenberg-Feller CLT** (Theorem A.2), we have that, as $N_u$ then $N_x$ approach infinity, $u_1^{(i)}\sum_{m=1}^{N_x}(\Lambda_i)_{m,m}\sum_{j,k=1}^{N_u}(\mathbf{W}_C)_{m,j}(\mathbf{W}_B)_{m,k}u_1^{(j)}u_1^{(k)}$ converges to a Gaussian distribution with mean 0 and variance $c\,\zeta(2)u_1^{(i)}\sigma_B\sigma_C\|\mathbf{u}_1\|_2^2$.

Accordingly, imposing the conditions for stability of $y_1^{(i)}$ requires that $\sigma_C \in \Theta(\sqrt{\frac{1}{N_xN_u}})$.

**Stability of $\mathbf{x}_l^{(i)}$ and $y_l^{(i)}$ for arbitrary $l \in [L]$.** For all $l \in [L]$, we have $\mathbf{x}_l^{(i)} = \sum_{m=0}^{l-1}(\mathbf{A}_l^{'(i)})^m\mathbf{B}_{l-m}^{'(i)}u_{l-m}^{(i)}$. First, observe that since $\mathbf{A}' = \text{diag}(a_1', \ldots, a_{N_x}')$ with $a_n' = e^{-\frac{1}{2}\tau_l^{(i)}}(\cos(\tau_l^{(i)}\pi n) + i\sin(\tau_l^{(i)}\pi n))$, we have that, for all complex vectors $\mathbf{v} \in \mathbb{C}^{N_x}$ it holds

that for any $m \in [L]$, $\|(\mathbf{A}')^m \mathbf{v}\|_2 = e^{-m\tau_l^{(i)}/2}\|\mathbf{v}\|_2$. Therefore, the operator $(\mathbf{A}'_i)^m$ does not change the width-scaling. Since, for any $\mathbf{u}_l$, setting $\sigma_B \in \Theta(\sqrt{\frac{N_x}{N_u}})$ yields $\left\|\mathbf{B}_l'^{(i)}\mathbf{u}_l\right\|_2 \in \Theta(\sqrt{N_x})$. Therefore, each term in the summation is of order $\Theta(\sqrt{N_x})$ and unless, for every $l$, the term $\mathbf{B}_l'^{(i)}\mathbf{u}_l$ perfectly cancels out with the terms before to affect the width scaling, we have that $\|\mathbf{x}_l^{(i)}\| \in \Theta(\sqrt{N_x})$. The same argument can be used to show the stability of $y_l^{(i)}$. $\qquad\square$

**Proposition 3.4 (Scale of the updates of hidden states $\Delta\mathbf{x}_1^{(i)}$ after 1 step of SGD).** *Under the ZOH discretization procedure, as $N_x$ then $N_u$ approach infinity, for every 1D SSM, the squared $l_2$-norm of the updates $\|\Delta\mathbf{x}_1^{(i)}\|_2$ of the hidden states after one step of SGD is a.s. scaled as $\|\Delta\mathbf{x}_1^{(i)}\|_2 \in \Theta\left(\eta_B \frac{1}{\sqrt{N_u}}\sigma_C \|\mathbf{u}_1\|_2^3 \zeta(4)^{\frac{1}{2}}\right)$, where $\zeta(4)$ denotes the Riemann zeta function at 4.*

***Proof of Proposition 3.4.*** Following condition ($\Delta$), the features of the 1D SSM evolve non-trivially with width, when the following condition is satisfied:

$$\text{If } |\Delta u_l^{(i)}| \in \Theta(1) \text{ then } \left\|\Delta\mathbf{x}_l^{(i)}\right\|_2 \in \Theta(\sqrt{N_x}) \text{ and } |\Delta y_l^{(i)}| \in \Theta(1).$$

### Scale of the updates $\Delta x_l^{(i)}$ after 1 SGD step.

First, note that, for the discretization step ZOH, we have

$$\bar{\mathbf{A}}^{(i)} = \tau^{(i)}\bar{\mathbf{A}}'^{(i)}\mathbf{A}'^{(i)} + (\tau^{(i)}\mathbf{A}'^{(i)}\mathbf{A}^{(i)-1} - (\mathbf{A}^{(i)})^{-2}(\mathbf{A}'^{(i)} - \mathbf{I}))(\bar{\mathbf{B}}_l'^{(i)}\mathbf{B}_l^{(i)\mathsf{T}} \odot \mathbf{I})$$

$$\bar{\mathbf{x}}_l^{(i)} = \bar{y}_l^{(i)}\mathbf{C}_l^{\mathsf{T}}$$

$$\bar{\mathbf{B}}_l = \sum_{i=1}^{N_u}(\mathbf{A}'^{(i)} - \mathbf{I})(\mathbf{A}^{(i)})^{-1}\bar{\mathbf{B}}_l'^{(i)}$$

$$\bar{\mathbf{A}}'^{(i)} = \sum_{l=1}^{L}(\bar{\mathbf{x}}_l^{(i)}\mathbf{x}_{l-1}^{(i)\mathsf{T}}) \odot \mathbf{I}$$

$$\mathrm{Diag}(\bar{\mathbf{A}}_{\log}^{(i)}) = \bar{\mathbf{A}}^{(i)}\mathbf{A}^{(i)}$$

$$\bar{\mathbf{B}}_l'^{(i)} = u_l^{(i)}\bar{\mathbf{x}}_l^{(i)}$$

$$\bar{\mathbf{W}}_B = \bar{\mathbf{B}}_l\mathbf{u}_l^{\mathsf{T}}$$

$$\bar{\mathbf{C}}_l = \sum_{i=1}^{N_u}\bar{y}_l^{(i)}\mathbf{x}_l^{(i)\mathsf{T}}$$

For any quantity $\cdot$, letting $\tilde{\cdot}$ denote the updated quantity after one step of SGD, we can write

$$\tilde{\mathbf{A}}^{(i)} = \mathrm{Diag}(\exp(\mathbf{a}_{\log}^{(i)} - \eta_a\bar{\mathbf{A}}_{\log}^{(i)})) = \mathbf{A}^{(i)}\big(\exp(-\eta_a\bar{\mathbf{A}}^{(i)}\mathbf{A}^{(i)}) \odot \mathbf{I}\big),$$

$$\tilde{\mathbf{B}}_l = \tilde{\mathbf{W}}_B\tilde{\mathbf{u}}_l = (\mathbf{W}_B - \eta_B\bar{\mathbf{B}}_l\mathbf{u}_l^T)(\mathbf{u}_l + \Delta\mathbf{u}_l).$$

For clarity, let us again begin by considering the scale of $\mathbf{x}_1^{(i)}$. We can follow the same argumentation as before to generalize the result to arbitrary $l \in [L]$. The updates $\Delta\mathbf{x}_1^{(i)}$ can therefore be computed as $\tilde{\mathbf{B}}_1'^{(i)}\tilde{u}_1^{(i)} - \mathbf{B}_1'^{(i)}u_1^{(i)}$, where $\tilde{\mathbf{B}}_1'^{(i)} = ((\tilde{\mathbf{A}}')^{(i)} - I)(\tilde{\mathbf{A}}^{(i)})^{-1}\tilde{\mathbf{B}}_1$ and $\tilde{\mathbf{A}}'^{(i)} = \exp(\tau\tilde{\mathbf{A}}^{(i)})$. Since $u_l^{(i)}$ and $\tilde{u}_l^{(i)}$ are assumed to be $\Theta(1)$ due to Assumption 3.1, the scale of $\Delta\mathbf{x}_1^{(i)}$ is determined by that of $\tilde{\mathbf{B}}_1'^{(i)}$ and $\tilde{\mathbf{B}}_1'^{(i)} - \mathbf{B}_1'^{(i)}$.

**Scaling of $\tilde{\mathbf{B}}_l'^{(i)}$.** Recall that $\tilde{\mathbf{B}}_1'^{(i)} = (\tilde{\mathbf{A}}'^{(i)} - \mathbf{I})(\tilde{\mathbf{A}}^{(i)})^{-1}\tilde{\mathbf{B}}_1 = (\tilde{\mathbf{A}}'^{(i)} - \mathbf{I})(\tilde{\mathbf{A}}^{(i)})^{-1}(\mathbf{W}_B + \Delta\mathbf{W}_B)(\mathbf{u}_1 + \Delta\mathbf{u}_1)$.

Let's begin by understanding the scale of $(\tilde{\mathbf{A}}'^{(i)} - \mathbf{I})(\tilde{\mathbf{A}}^{(i)})^{-1}(\Delta\mathbf{W}_B\mathbf{u}_1)$. Since $\Delta\mathbf{W}_B = -\eta_B\bar{\mathbf{B}}_1\mathbf{u}_1^T$, letting $\tilde{\mathbf{\Lambda}}_i = (\tilde{\mathbf{A}}'^{(i)} - \mathbf{I})(\tilde{\mathbf{A}}^{(i)})^{-1}$, we can express the $l_2$-norm of $(\tilde{\mathbf{A}}'^{(i)} - \mathbf{I})(\tilde{\mathbf{A}}^{(i)})^{-1}\Delta\mathbf{W}_B\mathbf{u}_1$ as follows:

$$\left\|(\tilde{\mathbf{A}}'^{(i)} - \mathbf{I})(\tilde{\mathbf{A}}^{(i)})^{-1}\Delta\mathbf{W}_B\mathbf{u}_1\right\|_2 = \left\|\mathbf{\Lambda}_i\eta_B\bar{\mathbf{B}}_1\mathbf{u}_1^T\mathbf{u}_1\right\|_2 = |\eta_B|\left\|\mathbf{\Lambda}_i\bar{\mathbf{B}}_1\right\|_2\|\mathbf{u}_1\|_2^2$$

Note that since $\tilde{\mathbf{\Lambda}}_i\Delta\mathbf{W}_B$ is a rank-one matrix, its spectral norm can be decomposed as

$$\|\mathbf{\Lambda}_i\Delta\mathbf{W}_B\mathbf{u}_1\|_* = |\eta_B|\left\|\mathbf{\Lambda}_i\bar{\mathbf{B}}_1\right\|_2\|\mathbf{u}_1\|_2$$

Therefore, since $\|\mathbf{u}_1\|_2 \in \Theta(\sqrt{N_u})$, if the spectral norm of $\mathbf{\Lambda}_i \Delta \mathbf{W}_B$ is scaled as $\Theta(\sqrt{\frac{N_x}{N_u}})$,

$$\left\| (\tilde{\mathbf{A}}'^{(i)} - \mathbf{I})(\tilde{\mathbf{A}}^{(i)})^{-1} \Delta \mathbf{W}_B \mathbf{u}_1 \right\|_2 \in \Theta(\sqrt{N_x}).$$

**Claim.** If $\|\Delta \mathbf{W}_B\|_* \in \Theta(\sqrt{\frac{N_x}{N_u}})$, then $\left\| (\tilde{\mathbf{A}}'^{(i)} - \mathbf{I})(\tilde{\mathbf{A}}^{(i)})^{-1} \Delta \mathbf{W}_B \mathbf{u}_1 \right\|_2 \in \Theta(\sqrt{N_x}).$

Essentially, this follows if $(\tilde{\mathbf{A}}'^{(i)} - \mathbf{I})(\tilde{\mathbf{A}}^{(i)})^{-1}$ does not affect the width-scaling of the spectral norm of $\Delta \mathbf{W}_B$, that is, $\|\mathbf{\Lambda}_i \Delta \mathbf{W}_B\|_* = \|\Delta \mathbf{W}_B\|_*$. This is equivalent to showing that $\left\| \tilde{\mathbf{\Lambda}}_i \bar{\mathbf{B}}_1 \right\|_2 = \left\| \bar{\mathbf{B}}_1 \right\|_2$.

Note the following Lemma which states that the scaling of $\|\tilde{\mathbf{\Lambda}}_i \bar{\mathbf{B}}_1\|_2$ is identical to that of $\|\mathbf{A}^{-1}\bar{\mathbf{B}}_1\|_2$.

**Lemma C.3.** *Let $Q_{i,1} = (\exp(-\eta_a \bar{\mathbf{A}}^{(i)} \mathbf{A}^{(i)})$. For any complex vector $\mathbf{v} \in \mathbb{C}^{N_x}$, the width-scaling of $\mathbf{v}$ remains invariant under the operator $(\tilde{\mathbf{A}}'^{(i)} - \mathbf{I})Q_{i,1}^{-1}$, that is, $\|(\tilde{\mathbf{A}}'^{(i)} - \mathbf{I})Q_{i,1}^{-1}\mathbf{v}\|_2 \in \Theta(\|\mathbf{v}\|_2)$.*

**Proof of Lemma C.3.** Recall that $\tilde{\mathbf{\Lambda}}_i = (\tilde{\mathbf{A}}'^{(i)} - \mathbf{I})\tilde{\mathbf{A}}_i^{-1}$, where $\tilde{\mathbf{A}}_i = \mathbf{A}^{(i)} Q_{i,1}$ and

$$\mathbf{Q}_{i,1} = \left( \exp(-\eta_a \bar{\mathbf{A}}^{(i)} \mathbf{A}^{(i)}) \odot \mathbf{I} \right) = \exp\left( -\eta_a ((\tau^{(i)} \mathbf{A}_l'^{(i)} - (\mathbf{A}^{(i)})^{-1}(\mathbf{A}_l'^{(i)} - \mathbf{I}))(\bar{\mathbf{B}}_1'^{(i)} \mathbf{B}_1^{(i)T} \odot \mathbf{I})) \right) \odot \mathbf{I}.$$

Since $[\mathbf{A}^{(i)}]_j = -(j+1)$, it's easy to see that, for any positive value of $\tau^{(i)}$, the eigenvalues of $\tau^{(i)} \mathbf{A}_l'^{(i)} - (\mathbf{A}^{(i)})^{-1}(\mathbf{A}_l'^{(i)} - \mathbf{I})$ are negative, in $\Theta(1)$, and converge to 0 from below. The expression evaluates to

$$\tau^{(i)} \mathbf{A}_l'^{(i)} - (\mathbf{A}^{(i)})^{-1}(\mathbf{A}_l'^{(i)} - \mathbf{I}) = \tau^{(i)} \exp(-\tau^{(i)}(j+1)) + \frac{\exp(-\tau^{(i)}(j+1)) - 1}{(j+1)}$$

Since $\bar{\mathbf{B}}_1'^{(i)} = u_1^{(i)} \bar{\mathbf{x}}_1^{(i)}$ and $\bar{\mathbf{x}}_1^{(i)} = \bar{y}_1^{(i)} C_1^T$,

$$[\bar{\mathbf{B}}_1'^{(i)} \mathbf{B}_1^{(i)T}]_{j,j} = \left( \sum_{m=1}^{N_u} [\mathbf{W}_B]_{j,m} u_1^{(m)} \right) \left( \sum_{m'=1}^{N_u} [\mathbf{W}_C]_{j,m'} u_1^{(m')} \right) u_1^{(i)} \bar{y}_1^{(i)}.$$

Due to Assumption 3.1, $\bar{y}_1^{(i)}$ is distributed i.i.d as $N_u \to \infty$ and $\bar{y}_1^{(i)} \in \Theta(\frac{1}{N_u})$ (Yang and Hu, 2021). A CLT argument reveals that, under the scaling $\sigma_B \in \Theta(\sqrt{\frac{N_x}{N_u}})$ and $\sigma_C \in \Theta(\sqrt{\frac{1}{N_x N_u}})$, the eigenvalues of $(\bar{\mathbf{B}}_1'^{(i)} \mathbf{B}_1^{(i)T} \odot \mathbf{I})$ are in $\Theta(\frac{1}{N_u})$. Lemma C.4 follows as a consequence of this result.

**Lemma C.4.** *As $N_u$ then $N_x$ approach infinity, $\|\tilde{\mathbf{A}}^{(i)} - \mathbf{A}^{(i)}\|_* \in \Theta(1)$ and $\|\tilde{\mathbf{A}}^{(i)}\|_* \in \Omega(1)$ if and only if $\eta_a \in \Theta(N_u)$.*

Accordingly, all the eigenvalues of $\mathbf{Q}_{i,1}$ are in $\Theta(1)$ and therefore, for any complex vector $\mathbf{v}$, $\|\mathbf{Q}_{i,1}\mathbf{v}\|_2 \in \Theta(\|\mathbf{v}\|_2)$. Together with the fact that the spectrum of $\mathbf{A}'^{(i)}$ is $\Theta(1)$, we have $\tilde{\mathbf{\Lambda}}_i' \in \Theta(1)$. Further note that the entries of $\tilde{\mathbf{A}}^{(i)}$ and consequently those of $\tilde{\mathbf{\Lambda}}_i$ are independent of each other since the entries $[\bar{\mathbf{B}}_1^{(i)} \mathbf{B}_1^{(i)T}]_{m,m}$ are independent for different $m \in [N_x]$.

Since $\bar{\mathbf{B}}_1 = \sum_{i=1}^{N_u} (\mathbf{A}'^{(i)} - \mathbf{I})(\mathbf{A}^{(i)})^{-1} \bar{\mathbf{B}}_1'^{(i)}$, it follows from Lemma C.3 that $\left\| \tilde{\mathbf{\Lambda}}_i \bar{\mathbf{B}}_1 \right\|_2 = \left\| \bar{\mathbf{B}}_1 \right\|_2$.

Therefore, for the correct scaling of the updates, $\eta_B$ should be scaled as $\Theta(\frac{N_x}{\sqrt{N_u}})$. Similar computations reveal that $\eta_C$ should be scaled as $\Theta(\frac{1}{\sqrt{N_x N_u}})$.

$\square$

**Proposition 3.5 (Scale of the updates of outputs $\Delta \mathbf{y}_1^{(i)}$ after 1 step of SGD).** *Under the ZOH discretization procedure, as $N_u$ then $N_x$ approach infinity, for every 1D SSM, the squared $l_2$-norm of the updates of the hidden states after one step of SGD scales as $|\Delta y_1^{(i)}|^2 \in \Theta\left( \eta_C \sigma_B^2 \sqrt{\frac{1}{N_u}} \|\mathbf{u}_1\|_2^4 \right)$.*

Before we present the proof of Proposition 3.5, we first provide the following Lemma.

**Lemma C.5.** *As $N_u \to \infty$, $\sum_{i=1}^{N_u} \bar{y}_1^{(i)} u_1^{(i)}$ converges to a normal distribution with mean $0$ and variance $N_u Var(\bar{y}_1^{(i)} u_1^{(i)})$. Note that since we assume that the rest of the network is scaled correctly (under $\mu P$), $\bar{y}_1^{(i)}$ is distributed i.i.d asymptotically and $\bar{y}_1^{(i)} \in \Theta(\frac{1}{N_u})$ (Yang and Hu, 2021). Therefore $\sum_{i=1}^{N_u} \bar{y}_1^{(i)} u_1^{(i)} \in \Theta(\frac{1}{\sqrt{N_u}})$.*

*Proof.* For $i \in [N_u]$, the output of the $1D$ SSM is given by

$$y_1^{(i)} = \mathbf{C}_1 \mathbf{x}_1^{(i)} \quad \text{where} \quad \mathbf{C}_1 = (\mathbf{W}_C \mathbf{u}_1)^T \quad \text{and} \quad \mathbf{x}_1^{(i)} = \mathbf{B}_1'^{(i)} u_1^{(i)} = (\mathbf{A}'^{(i)} - \mathbf{I})\mathbf{A}^{(i)^{-1}} \mathbf{W}_B \mathbf{u}_1. \quad (20)$$

Using the notation $\tilde{\cdot}$ to denote the updated quantity $\cdot$ after 1 step of SGD,

$$\tilde{y}_1^{(i)} = \tilde{\mathbf{C}}_1 \tilde{\mathbf{x}}_1^{(i)} = (\tilde{\mathbf{W}}_C \tilde{\mathbf{u}}_1)^T (\tilde{\mathbf{\Lambda}}_i \tilde{\mathbf{W}}_B \tilde{\mathbf{u}}_1) \tilde{u}_1^{(i)}. \quad (21)$$

The updated quantities $\tilde{\mathbf{W}}_B$ and $\tilde{\mathbf{W}}_C$ are derived as

$$\tilde{\mathbf{W}}_B = \mathbf{W}_B + \Delta \mathbf{W}_B, \quad \tilde{\mathbf{W}}_C = \mathbf{W}_C + \Delta \mathbf{W}_C \quad \text{with} \quad \Delta \mathbf{W}_B = -\eta_B \bar{\mathbf{W}}_B, \quad \Delta \mathbf{W}_C = -\eta_C \bar{\mathbf{W}}_C,$$

where $\bar{\mathbf{W}}_B$ and $\bar{\mathbf{W}}_C$ denote the gradient of the loss with respect to the quantities $\mathbf{W}_B$ and $\mathbf{W}_C$ respectively and can be computed according to

$$\bar{\mathbf{W}}_B = \sum_{i=1}^{N_u} \bar{y}_1^{(i)} u_1^{(i)} \mathbf{\Lambda}_i \mathbf{W}_B \mathbf{u}_1 \mathbf{u}_1^T \quad \text{and} \quad \bar{\mathbf{W}}_C = \sum_{i=1}^{N_u} \bar{y}_1^{(i)} u_1^{(i)} \mathbf{\Lambda}_i \mathbf{C}_1^T \mathbf{u}_1^T. \quad (22)$$

Also, $\tilde{\mathbf{\Lambda}}_i = (\mathbf{A}^{\tilde{'}(i)} - \mathbf{I})\mathbf{A}^{\tilde{}(i)^{-1}}$, where $\mathbf{A}^{\tilde{}(i)} = \text{Diag}(\exp(\mathbf{a}_i^{\log} - \eta_a \bar{\mathbf{A}}_{\log}^{(i)})) = \mathbf{A}^{(i)} \mathbf{Q}_i$, with $\mathbf{Q}_i = (\exp(-\eta_a \bar{\mathbf{A}}^{(i)} \mathbf{A}^{(i)}) \odot \mathbf{I})$ and

$$\bar{\mathbf{A}}^{(i)} = (\tau^{(i)} \mathbf{A}'^{(i)} (\mathbf{A}^{(i)})^{-1} - (\mathbf{A}^{(i)})^{-2} (\mathbf{A}'^{(i)} - \mathbf{I}))(\bar{\mathbf{B}}_1'^{(i)} \mathbf{B}_1^{(i)\mathsf{T}} \odot \mathbf{I}).$$

Therefore $\tilde{y}_1^{(i)}$ can be represented as follows

$$\tilde{y}_1^{(i)} = \left[(\mathbf{W}_C + \Delta \mathbf{W}_C)(\mathbf{u}_1 + \Delta \mathbf{u}_1)\right]^T \left[\tilde{\mathbf{\Lambda}}_i (\mathbf{W}_B + \Delta \mathbf{W}_B)(\mathbf{u}_1 + \Delta \mathbf{u}_1)\right](u_1^{(i)} + \Delta u_1^{(i)}), \quad (23)$$

and the updates $\tilde{y}_1^{(i)} - y_1^{(i)}$ can be expressed as

$$\tilde{y}_1^{(i)} - y_1^{(i)} = (\mathbf{W}_C \mathbf{u}_1)^T \tilde{\mathbf{\Lambda}}_i \mathbf{W}_B \mathbf{u}_1 u_1^{(i)} - (\mathbf{W}_C \mathbf{u}_1)^T \mathbf{\Lambda}_i \mathbf{W}_B \mathbf{u}_1 u_1^{(i)} + \mathcal{M},$$

where $\mathcal{M} = \left[(\mathbf{W}_C + \Delta \mathbf{W}_C)(\mathbf{u}_1 + \Delta \mathbf{u}_1)\right]^T \left[\tilde{\mathbf{\Lambda}}_i (\mathbf{W}_B + \Delta \mathbf{W}_B)(\mathbf{u}_1 + \Delta \mathbf{u}_1)\right](u_1^{(i)} + \Delta u_1^{(i)}) - (\mathbf{W}_C \mathbf{u}_1)^T \tilde{\mathbf{\Lambda}}_i \mathbf{W}_B \mathbf{u}_1 u_1^{(i)} - (\mathbf{W}_C \mathbf{u}_1)^T \mathbf{\Lambda}_i \mathbf{W}_B \mathbf{u}_1 u_1^{(i)}$ represents the remainder of the terms in (23).

First, let us consider the scaling of the update term without remainder term,

$$(\mathbf{W}_C \mathbf{u}_1)^T (\tilde{\mathbf{\Lambda}}_i - \mathbf{\Lambda}_i) \mathbf{W}_B \mathbf{u}_1 u_1^{(i)} = \sum_{m=1}^{N_x} \mathcal{K}_{m,m} \sum_{m',m''=1}^{N_u} \mathbf{W}_{Cm,m'} \mathbf{W}_{Bm,m''} \mathbf{u}_1^{(m')} \mathbf{u}_1^{(m'')},$$

where $\mathcal{K} = (\tilde{\mathbf{\Lambda}}_i - \mathbf{\Lambda}_i)$.

Letting $\mathbf{v}_m = \sum_{m',m''=1}^{N_u} \mathbf{W}_{Cm,m'} \mathbf{W}_{Bm,m''} \mathbf{u}_1^{(m')} \mathbf{u}_1^{(m'')}$, as $N_u \to \infty$, $\{\mathbf{v}_m\}_{m=1}^{N_x}$ are independent and are distributed normally with mean $0$ and variance $\sigma_B^2 \sigma_C^2 \|\mathbf{u}_1\|^4$ by an application of Lindenberg-Feller Central Limit Theorem.

**Independence of entries of $\mathcal{K}$.** Next, we show that the entries of the diagonal matrix $\mathcal{K}$ are independent of each other.

Observe that

$$\mathcal{K} = (\tilde{\mathbf{\Lambda}}_i - \mathbf{\Lambda}_i) = (\mathbf{A}^{\tilde{'}(i)} - \mathbf{I})\mathbf{A}^{\tilde{}(i)^{-1}} - (\mathbf{A}'^{(i)} - \mathbf{I})(\mathbf{A}^{(i)})^{-1},$$

where $\mathbf{A}^{\tilde{(i)}} = \mathrm{Diag}(\exp{(\mathbf{a}_i^{\log} - \eta_a \bar{\mathbf{A}}_{\log}^{(i)})}) = \mathbf{A}^{(i)}\mathbf{Q}_i$, with $\mathbf{Q}_i = \left(\exp{(-\eta_a \bar{\mathbf{A}}^{(i)}\mathbf{A}^{(i)})} \odot \mathbf{I}\right)$ and

$$\bar{\mathbf{A}}^{(i)} = (\tau^{(i)}\mathbf{A}'^{(i)}(\mathbf{A}^{(i)})^{-1} - (\mathbf{A}^{(i)})^{-2}(\mathbf{A}'^{(i)} - \mathbf{I}))(\bar{\mathbf{B}}_1'^{(i)}\mathbf{B}_1^{(i)\mathsf{T}} \odot \mathbf{I}).$$

Since the entries of $\mathbf{A}$ are deterministic, and we assume that $\tau^{(i)}$ is some fixed deterministic scalar, the entries of $\mathcal{K}$ are independent of each other iff the entries of $(\bar{\mathbf{B}}_1'^{(i)}\mathbf{B}_1^{(i)\mathsf{T}} \odot \mathbf{I})$ are independent. Now consider

$$[\bar{\mathbf{B}}_1'^{(i)}\mathbf{B}_1^{(i)T}]_{m,m} = \sum_{m',m''=1}^{N_u} \mathbf{W}_{B\,m,m'}\mathbf{W}_{C\,m,m''}\mathbf{u}_1^{(m')}\mathbf{u}_1^{(m'')}u_1^{(i)}\bar{y}_1^{(i)}.$$

Since $\mathbf{W}_B$ and $\mathbf{W}_C$ are Gaussian matrices with i.i.d entries and the entries of $\mathbf{u}_1$ are assumed to be i.i.d, the entries $[\bar{\mathbf{B}}_1'^{(i)}\mathbf{B}_1^{(i)T}]_{m,m}$ are independent for different $m \in [N_x]$.

Furthermore, following the same line of argumentation as in Lemma C.3, the matrix $\mathcal{K}$ can be represented as $\mathcal{K} = \mathcal{K}'\mathbf{A}^{-1}$, where the entries of $\mathcal{K}'$ are $\Theta(1)$ and accordingly for any vector the scaling of $v$ remains invariant under the operator $\mathcal{K}'$.

To compute the scale of $\sum_{m=1}^{N_x} \mathcal{K}_{m,m}\mathbf{v}_m$, we can therefore instead consider the scaling of $\sum_{m=1}^{N_x} \mathbf{A}_{m,m}^{-1}\mathbf{v}_m$ where $\{\mathbf{v}_m\}_{m=1}^{N_x}$ are independent and are distributed normally with mean 0 and variance $\sigma_B^2\sigma_C^2\|\mathbf{u}_1\|^4$.

As $N_x \to \infty$, $\sum_{m=1}^{N_x} \mathbf{A}_{m,m}^{-1}\mathbf{v}_m$ converges to a normal distribution with mean 0 and variance $c\,\zeta(2)\sigma_B^2\sigma_C^2\|\mathbf{u}_1\|^4$, where $\zeta(2)$ denotes the Riemann zeta function at 2 and $c\,\zeta(2) > 0$ evaluates to a width-independent constant.

When $\sigma_B \in \Theta(\sqrt{\frac{N_x}{N_u}})$ and $\sigma_C \in \Theta(\frac{1}{\sqrt{N_x N_u}})$, we get $(\mathbf{W}_C\mathbf{u}_1)^T(\tilde{\mathbf{\Lambda}}_i - \mathbf{\Lambda}_i)\mathbf{W}_B\mathbf{u}_1 u_1^{(i)} \in \Theta(1)$.

**Scale of $\mathcal{M}$.**

Note that since $u_1^{(i)}$ and $\Delta u_1^{(i)}$ are scalars and in order 1, the scale of $\mathcal{M}$ is determined by that of $[(\mathbf{W}_C + \Delta\mathbf{W}_C)(\mathbf{u}_1 + \Delta\mathbf{u}_1)]^T[\tilde{\mathbf{\Lambda}}_i(\mathbf{W}_B + \Delta\mathbf{W}_B)(\mathbf{u}_1 + \Delta\mathbf{u}_1)]$.

Recall that both $\mathbf{u}_1$ and $\Delta\mathbf{u}_1$ are assumed have i.i.d coordinates and have the scaling $\|\mathbf{u}_1\| \in \Theta(\sqrt{N_x})$ and $\|\Delta\mathbf{u}_1\| \in \Theta(\sqrt{N_x})$. Therefore, we only need to consider the scaling of

$$[(\mathbf{W}_C + \Delta\mathbf{W}_C)\mathbf{u}_1]^T[\tilde{\mathbf{\Lambda}}_i(\mathbf{W}_B + \Delta\mathbf{W}_B)(\mathbf{u}_1)].$$

It is easy to verify that the scaling of $(\mathbf{W}_C\mathbf{u}_1)^T(\tilde{\mathbf{\Lambda}}_i - \mathbf{\Lambda}_i)\mathbf{W}_B\mathbf{u}_1$ is the same as that of $(\mathbf{W}_C\mathbf{u}_1)^T(\tilde{\mathbf{\Lambda}}_i)\mathbf{W}_B\mathbf{u}_1 u_1^{(i)}$ following the same arguments as above.

**Scale of $(\Delta\mathbf{W}_C\mathbf{u}_1)^T(\tilde{\mathbf{\Lambda}}_i)\mathbf{W}_B\mathbf{u}_1 u_1^{(i)}$.**

Recall that the updates $\Delta\mathbf{W}_B$ and $\Delta\mathbf{W}_C$ are defined as

$$\Delta\mathbf{W}_B = -\eta_B\bar{\mathbf{W}}_B \quad \text{and} \quad \Delta\mathbf{W}_C = -\eta_C\bar{\mathbf{W}}_C,$$

where $\bar{\mathbf{W}}_B$ and $\bar{\mathbf{W}}_C$ denote the gradient of the loss with respect to the quantities $\mathbf{W}_B$ and $\mathbf{W}_C$ respectively and computed as

$$\bar{\mathbf{W}}_B = \sum_{i=1}^{N_u} \bar{y}_1^{(i)}u_1^{(i)}\mathbf{\Lambda}_i\mathbf{W}_B\mathbf{u}_1\mathbf{u}_1^T \quad \text{and} \quad \bar{\mathbf{W}}_C = \sum_{i=1}^{N_u} \bar{y}_1^{(i)}u_1^{(i)}\mathbf{\Lambda}_i\mathbf{C}_1^T\mathbf{u}_1^T. \tag{24}$$

So, $\Delta\mathbf{W}_C\mathbf{u}_1 = -\eta_C\sum_{i=1}^{N_u}\bar{y}_1^{(i)}u_1^{(i)}\mathbf{\Lambda}_i\mathbf{W}_B\mathbf{u}_1\mathbf{u}_1^T\mathbf{u}_1 = -\eta_C\|\mathbf{u}_1\|^2\mathbf{\Lambda}_i\mathbf{W}_B\mathbf{u}_1\sum_{i=1}^{N_u}\bar{y}_1^{(i)}u_1^{(i)}$ ($\mathbf{\Lambda}_i$ is identical for all $i \in [N_u]$).

Therefore, letting $\Xi = -\eta_C\|\mathbf{u}_1\|^2\sum_{i=1}^{N_u}\bar{y}_1^{(i)}u_1^{(i)}$, we get

$$(\Delta\mathbf{W}_C\mathbf{u}_1)^T(\tilde{\boldsymbol{\Lambda}}_i\mathbf{W}_B\mathbf{u}_1) = \Xi(\boldsymbol{\Lambda}_i\mathbf{W}_B\mathbf{u}_1)^T(\tilde{\boldsymbol{\Lambda}}_i\mathbf{W}_B\mathbf{u}_1)$$

$$= \Xi\sum_{m=1}^{N_x}\boldsymbol{\Lambda}_{im,m}\tilde{\boldsymbol{\Lambda}}_{im,m}\sum_{m',m''=1}^{N_u}\mathbf{W}_{Bm,m'}\mathbf{W}_{Bm,m''}\mathbf{u}_1^{(m')}\mathbf{u}_1^{(m'')}.$$

Letting $\mathbf{w}_m = \sum_{m',m''=1}^{N_u}\mathbf{W}_{Bm,m'}\mathbf{W}_{Bm,m''}\mathbf{u}_1^{(m')}\mathbf{u}_1^{(m'')}$, it is easy to verify that as $N_x \to \infty$

each $v_m$ is distributed normally with mean $\sigma_B^2\|\mathbf{u}_1\|^2$ and variance $\sigma_B^4\sum_{m'\neq m''=1}^{N_u}\mathbf{u}_1^{(m')^2}\mathbf{u}_1^{(m'')^2}$.

Furthermore $\{\mathbf{w}_m\}_{m=1}^{N_x}$ are independent.

Therefore, due to Kolmogorov's SLLN, as $N_x \to \infty$, it holds that

$$\sum_{m=1}^{N_x}\boldsymbol{\Lambda}_{im,m}\tilde{\boldsymbol{\Lambda}}_{im,m}\mathbf{w}_B \xrightarrow{\text{a.s}} \sigma_B^2\|\mathbf{u}_1\|^2.$$

Combining this with the result from Lemma C.5, we conclude that as $N_u \to \infty$ then $N_x \to \infty$, $(\Delta\mathbf{W}_C\mathbf{u}_1)^T(\tilde{\boldsymbol{\Lambda}}_i\mathbf{W}_B\mathbf{u}_1) \in \Theta(\eta_C\sigma_B^2\|\mathbf{u}_1\|^4\frac{1}{\sqrt{N_u}})$.

Therefore, for the updates $\Delta y_1^{(i)}$ to be of order 1, $\eta_C$ must be scaled as $\Theta(\frac{1}{N_x\sqrt{N_u}})$.

$\square$

**Scale of $(\Delta\mathbf{W}_C\mathbf{u}_1)^T(\tilde{\boldsymbol{\Lambda}}_i)\Delta\mathbf{W}_B\mathbf{u}_1 u_1^{(i)}$.**

Recall that the updates $\Delta\mathbf{W}_B$ and $\Delta\mathbf{W}_C$ are defined as

$$\Delta\mathbf{W}_B = -\eta_B\bar{\mathbf{W}}_B \quad \text{and} \quad \Delta\mathbf{W}_C = -\eta_C\bar{\mathbf{W}}_C,$$

where $\bar{\mathbf{W}}_B$ and $\bar{\mathbf{W}}_C$ denote the gradient of the loss with respect to the quantities $\mathbf{W}_B$ and $\mathbf{W}_C$ respectively and computed as

$$\bar{\mathbf{W}}_C = \sum_{i=1}^{N_u}\bar{y}_1^{(i)}u_1^{(i)}\boldsymbol{\Lambda}_i\mathbf{W}_B\mathbf{u}_1\mathbf{u}_1^T \quad \text{and} \quad \bar{\mathbf{W}}_B = \sum_{i=1}^{N_u}\bar{y}_1^{(i)}u_1^{(i)}\boldsymbol{\Lambda}_i\mathbf{C}_1^T\mathbf{u}_1^T. \tag{25}$$

Hence, we get $\Delta\mathbf{W}_C\mathbf{u}_1 = -\eta_C\sum_{i=1}^{N_u}\bar{y}_1^{(i)}u_1^{(i)}\boldsymbol{\Lambda}_i\mathbf{W}_B\mathbf{u}_1\mathbf{u}_1^T\mathbf{u}_1 = -\eta_C\|\mathbf{u}_1\|^2\boldsymbol{\Lambda}_i\mathbf{W}_B\mathbf{u}_1\sum_{i=1}^{N_u}\bar{y}_1^{(i)}u_1^{(i)}$

($\boldsymbol{\Lambda}_i$ is identical for all $i \in [N_u]$) and $\Delta\mathbf{W}_B\mathbf{u}_1 = \boldsymbol{\Lambda}_i\mathbf{W}_C\mathbf{u}_1\mathbf{u}_1^T\sum_{i=1}^{N_u}\bar{y}_1^{(i)}u_1^{(i)}$, which yields

$$(\Delta\mathbf{W}_C\mathbf{u}_1)^T(\tilde{\boldsymbol{\Lambda}}_i\Delta\mathbf{W}_B\mathbf{u}_1) = (\sum_{i=1}^{N_u}\bar{y}_1^{(i)}u_1^{(i)})^2\eta_C\eta_B\|\mathbf{u}_1\|^4(\boldsymbol{\Lambda}_i\mathbf{W}_B\mathbf{u}_1)^T(\tilde{\boldsymbol{\Lambda}}_i\mathbf{W}_C\mathbf{u}_1).$$

Following the same line of argumentation as before, we obtain that as $N_u \to \infty$ then $N_x \to \infty$, $(\Delta\mathbf{W}_C\mathbf{u}_1)^T(\tilde{\boldsymbol{\Lambda}}_i\Delta\mathbf{W}_B\mathbf{u}_1) \in \Theta(\sigma_B\sigma_C\eta_B\eta_C\|\mathbf{u}_1\|^6\frac{1}{N_u})$.

Substituting $\sigma_B \in \Theta(\sqrt{\frac{N_x}{N_u}}), \sigma_C \in \Theta(\frac{1}{\sqrt{N_xN_u}})$, $\eta_B \in \Theta(\frac{N_x}{\sqrt{N_u}})$ yields the same scaling: $\eta_C \in \Theta(\frac{1}{N_x\sqrt{N_u}})$.

**Scale of $(\mathbf{W}_C\mathbf{u}_1)^T(\tilde{\boldsymbol{\Lambda}}_i)\Delta\mathbf{W}_B\mathbf{u}_1 u_1^{(i)}$.** Following the same steps as for the previous terms, we obtain

$(\mathbf{W}_C\mathbf{u}_1)^T(\tilde{\boldsymbol{\Lambda}}_i)\Delta\mathbf{W}_B\mathbf{u}_1 u_1^{(i)} \in \Theta(\eta_B\frac{1}{\sqrt{N_u}}\sigma_C^2\|\mathbf{u}_1\|^2)$. This term vanishes under the scaling $\eta_B \in \Theta(\frac{N_x}{\sqrt{N_u}})$ and $\sigma_C \in \Theta(\frac{1}{\sqrt{N_xN_u}})$.

## C.3 Time-invariant S4 recurrent layer

The S4 recurrent layer $\mathbf{y}_{1:L} = f_{S4}(\mathbf{u}_{1:L}; \mathbf{w} = \{\mathbf{B}, \mathbf{C}\})$ is a sequence to sequence mapping:

$$\mathbf{x}_l = \mathbf{A}'\mathbf{x}_{l-1} + \mathbf{B}'\mathbf{u}_l \tag{26}$$

$$\mathbf{y}_l = \mathsf{R}[\mathbf{C}'\mathbf{x}_l] \tag{27}$$

for $l = 1, \ldots, L$ with $\mathbf{x}_0 = 0$, which is a discretization of the continuous-time SSM:

$$\frac{d}{dt}\mathbf{x}_t = \mathbf{A}\mathbf{x}_t + \mathbf{B}\mathbf{u}_t \tag{28}$$

$$\mathbf{y}_t = \mathsf{R}[\mathbf{C}\mathbf{x}_t] \tag{29}$$

where $\mathsf{R}(\cdot)$ gives the real part of a complex vector, $\mathbf{A} = \mathrm{diag}(\mathbf{a})$ and $\mathbf{a} \in \mathbb{C}^{N_x}$, $\mathbf{B} \in \mathbb{C}^{N_x \times N_u}$, $\mathbf{C} \in \mathbb{C}^{N_y \times N_x}$.

The Zero-Order-Hold (ZOH) discretization method gives that:

$$\mathbf{A}' = \exp(\tau_l^{(i)} \cdot \mathbf{A}) \tag{30}$$

$$\mathbf{B}' = (\mathbf{A}' - \mathbf{I})\mathbf{A}^{-1}\mathbf{B} \tag{31}$$

$$\mathbf{C}' = \mathbf{C} \tag{32}$$

To model long-range dependency, S4 advocates for HiPPO theory. While our results hold for every Hippo initialization matrix, here, we show the result for S4D-Lin and set $\mathbf{a}_n = -\frac{1}{2} + i\pi n$. Note that $\mathbf{A}$ is not parameterized and will not be trained. It has been observed that complex values are not essential and S4D-real with $\mathbf{a}_n = -(n+1)$ may perform similarly well.

The backward pass for $f_{S4}$ is as follows:

$$\bar{\mathbf{x}}_l = \mathbf{C}'^{\mathsf{T}}\bar{\mathbf{y}}_l + \mathbf{A}'^{\mathsf{T}}\bar{\mathbf{x}}_{l+1} \tag{33}$$

$$\bar{\mathbf{C}}' = \sum_{l=1}^{L} \bar{\mathbf{y}}_l \mathbf{x}_l^{\mathsf{T}} \tag{34}$$

$$\bar{\mathbf{B}}' = \sum_{l=1}^{L} \bar{\mathbf{x}}_l \mathbf{u}_l^{\mathsf{T}} \tag{35}$$

$$\bar{\mathbf{u}}_l = \mathbf{B}'^{\mathsf{T}}\bar{\mathbf{x}}_l \tag{36}$$

Since we want all learnable weight matrices to learn features, we split each S4 layer into two sublayers and demand them both to admit feature learning. The first one being the state space equation (26) and the second one being the decoder (27).

**Claim C.1 (Scale of hidden states $\mathbf{x}_1$ in S4 at initialization).** *Under the ZOH discretization procedure, as $N_x$ and $N_u$ approach infinity with $N_u/N_x \in \Theta(1)$, for any $i \in [N_u]$, the squared $l_2$-norm of the hidden states $\left\|\mathbf{x}_1^{(i)}\right\|_2^2$ is almost surely scaled as $\left\|\mathbf{x}_1^{(i)}\right\|_2^2 \in \Theta\left(\zeta(2)\sigma_B^2 \|\mathbf{u}\|^2\right)$, where $\zeta(2)$ is the Riemann zeta function at 2.*

*Proof.* **Stability at initialization.**

First, we show that the size of $\mathbf{x}_1 = \mathbf{B}'\mathbf{u}_1$ at initialization is $\Theta(\sqrt{N_x})$. By definition,

$$\|\mathbf{x}_1\|_2 = \|\mathbf{B}'\mathbf{u}_1\|_2 = \|\Lambda\mathbf{B}\mathbf{u}_1\|_2,$$

where $\Lambda = (\mathbf{A}' - \mathbf{I})\mathbf{A}^{-1}$. Then, writing $\mathbf{u} = \mathbf{u}_1$ for readability, observe that

$$\|\mathbf{B}'\mathbf{u}\|_2^2 = \sum_{i=1}^{N_x} \Lambda_i^2 \left(\sum_{j=1}^{N_u} B_{i,j}u_j\right)^2$$

is a sum of $N_x$ independent but not identically distributed random variables. However, we show that they satisfy the Kolmogorov condition and therefore the sum behaves according to the strong law

of large numbers. Intuitively, this is allowed by the polynomial decay of eigenvalues of $\Lambda$ and the correct scaling of the inputs to the S4 layer.

**Verifying that the sequence satisfies Kolmogorov condition.**

For any $i \in [N_x]$,

$$Var(\Lambda_i^2 (\sum_{j=1}^{N_u} B_{i,j} u_j)^2) = E\big[\Lambda_i^4 (\sum_{j=1}^{N_u} B_{i,j} u_j)^4\big] - E\big[\Lambda_i^2 (\sum_{j=1}^{N_u} B_{i,j} u_j)^2\big]^2$$

$$\leq E\big[\Lambda_i^4 (\sum_{j=1}^{N_u} B_{i,j} u_j)^4\big]$$

$$= \Lambda_i^4 E\big[\sum_{j,j',k,k'=1}^{N_u} B_{i,j} B_{i,j'} B_{i,k} B_{i,k'} u_j u_{j'}, u_k, u_{k'}\big]$$

$$= \Lambda_i^4 \big(E\big[\sum_{j=1}^{N_u} B_{i,j}^4 u_j^4\big] + E\big[\sum_{j\neq j'=1}^{N_u} B_{i,j}^2 B_{i,j'}^2 u_j^2 u_{j'}^2\big]\big)$$

$$= \Lambda_i^4 \big(3\sigma_B^4 \sum_{j=1}^{N_u} u_j^4 + \sigma_B^4 \sum_{j\neq j'=1}^{N_u} u_j^2 u_{j'}^2\big)$$

$$\leq 3\sigma_B^4 \Lambda_i^4 \|\mathbf{u}\|_2^4$$

$$\sum_{i=1}^{N_x} Var(\Lambda_i^2 (\sum_{j=1}^{N_u} B_{i,j} u_j)^2) \leq 3\sigma_B^4 \|\mathbf{u}\|_2^4 \sum_{i=1}^{N_x} \Lambda_i^4$$

$$= 3\sigma_B^4 \|\mathbf{u}\|_2^4 \sum_{i=1}^{N_x} \frac{(\exp(-(i+1)\cdot\tau_l^{(i)}) - 1)^4}{(i+1)^4}$$

$$\leq 3\sigma_B^4 \|\mathbf{u}\|_2^4 \sum_{i=1}^{N_x} \frac{1}{(i+1)^4}$$

$$= 3\sigma_B^4 \|\mathbf{u}\|_2^4 \zeta(4), \tag{37}$$

where $\zeta(4)$ denotes the Riemann zeta function at $4$ for which a closed form is given by $\frac{|\mathcal{B}_4|(2\pi)^4}{2\cdot 4!}$. Note that, due to the rapid decay of $\exp(-(i+1)\cdot\tau_l^{(i)}) \to 0$ with $i \to \infty$, the zeta function also lower bounds the spectral sums $\sum_{i=1}^{N_x} \Lambda_i^2 \geq c \cdot \zeta(2)$ and $\sum_{i=1}^{N_x} \Lambda_i^4 \geq c \cdot \zeta(4)$ for some width-independent constant $c \in (0,1)$ for $N_x$ large enough.

By assumption, the inputs to the S4 layer are scaled such that $\|\mathbf{u}\| \in \Theta(\sqrt{N_x})$ and therefore, as long as $\sigma_B^2 \in \mathcal{O}(1)$ the sequence of random variables satisfies the Kolmogorov condition.

**Limiting behavior of $\|\mathbf{B}'\mathbf{u}\|_2^2$.**

Applying SLLN, we have

$$\sum_{i=1}^{N_x} \Lambda_i^2 (\sum_{j=1}^{N_u} B_{i,j} u_j)^2 \longrightarrow \sum_{i=1}^{N_x} \Lambda_i^2 E\big[(\sum_{j=1}^{N_u} B_{i,j} u_j)^2\big]$$

$$= \sum_{i=1}^{N_x} \Lambda_i^2 E\big[\sum_{j=1}^{N_u} B_{i,j}^2 u_j^2\big]$$

$$= \sigma_B^2 \sum_{i=1}^{N_x} \Lambda_i^2 \sum_{j=1}^{N_u} u_j^2$$

$$= c\zeta(2)\sigma_B^2 \|\mathbf{u}\|^2,$$

where $\zeta(2)$ denotes the Riemann zeta function at 2 and equates to $\pi^2/6$, and $c \in (0,1)$ is some width-independent constant.

**Spectral scaling does not yield the correct scale of initialization for $\sigma_B$.**

For spectral scaling conditions to yield the right scaling of the initialization variance, it is crucial that the following condition holds:

$$\|\mathbf{B}'\mathbf{u}\|_2 \in \Theta(\|\mathbf{B}'\|_* \|\mathbf{u}\|_2).$$

Since the spectrum of $(\mathbf{A}' - \mathbf{I})\mathbf{A}^{-1}$ is less than 1, an upper bound on the spectral norm of $\mathbf{B}' = (\mathbf{A}' - \mathbf{I})\mathbf{A}^{-1}\mathbf{B}$ can be found in (Vershynin, 2011) and is given by $C(\sqrt{N_x} + \sqrt{N_u})$ for some width-independent constant $C$.

A matching lower bound can be easily found by noting that the spectral norm is lower bounded by the maximal row and column norm.

$$
\begin{aligned}
\|\mathbf{B}'\|_* &\geq \max\left\{\max_i \|B'_{i:}\|_2, \max_j \|B'_{:j}\|_2\right\} \\
&\geq \max_i \|B'_{i:}\|_2 \\
&= \max_i |\Lambda_i| \|B_{i:}\|_2 \\
&= \max_i |\Lambda_i| (\sum_{j=1}^{N_u} B_{i,j}^2)^{1/2} \\
&\approx \max_i |\Lambda_i| \sqrt{N_u}\sigma_B \\
&\geq \sqrt{N_u}\sigma_B
\end{aligned}
$$

Therefore $\frac{\|\mathbf{B}'\mathbf{u}\|_2}{(\|\mathbf{B}'\|_* \|\mathbf{u}\|_2)} \in \Theta(\frac{1}{\sqrt{N_u}})$

**Correct scale of $\sigma_B$ for stability at initialization.**

We know that $\|\mathbf{B}'\mathbf{u}\|_2 \in \Theta(\sigma_B \|\mathbf{u}\|_2)$ and since $\|\mathbf{u}\|^2 \in \Theta(\sqrt{N_u})$ (since we assumed stablity of activations in the previous layer), imposing stability of $\|\mathbf{B}'\mathbf{u}\|_2$ yields the following scaling: $\sigma_B \in \Theta(\sqrt{\frac{N_x}{N_u}})$.

**Stability of $\mathbf{x}_l$ for arbitrary $l \in [L]$.**

For all $l \in [L]$, we have $\mathbf{x}_l = \sum_{m=1}^{l-1}(\mathbf{A}')^m\mathbf{B}'\mathbf{u}_{l-m} = \mathbf{A}'(\sum_{m=1}^{l-2}(\mathbf{A}')^m\mathbf{B}'\mathbf{u}_{l-m}) + \mathbf{B}'\mathbf{u}_l$. First, observe that since $\mathbf{A}' = \text{diag}(\mathbf{a}'_1, \ldots, \mathbf{a}'_{N_x})$ with $\mathbf{a}'_n = e^{-\frac{1}{2}\tau_l^{(i)}}(\cos(\tau_l^{(i)}\pi n) + i\sin(\tau_l^{(i)}\pi n))$, we have that, for all complex vectors $\mathbf{v} \in \mathbb{C}^{N_x}$ it holds that for any $m \in [L]$, $\|(\mathbf{A}')^m\mathbf{v}\|_2 = e^{-m\tau_l^{(i)}/2}\|\mathbf{v}\|_2$. Therefore, the operator $(\mathbf{A}')^m$ does not change the width-scaling. We showed earlier that for any $\mathbf{u}_l$, setting $\sigma_B \in \Theta(\sqrt{\frac{N_x}{N_u}})$ yields $\|\mathbf{B}'\mathbf{u}_l\|_2 \in \Theta(\sqrt{N_x})$. Therefore, each term in the summation is of order $\Theta(\sqrt{N_x})$ and unless, for every $l$, the term $\mathbf{B}'\mathbf{u}_l$ perfectly cancels out with the terms before to affect the width scaling, we have that $\mathbf{x}_l \in \Theta(\sqrt{N_x})$ for all $l \in [L]$.

$\square$

**Claim C.2** (Scale of output $y^{(i)}$ of S4 at initialization). *Under the ZOH discretization procedure as $N_x$ and $N_u$ approach infinity with $N_u/N_x \in \Theta(1)$, for any $i \in [N_u]$, the output $y_1^{(i)}$ converges in distribution to a Gaussian with mean 0 and standard deviation $C\sigma_B\sigma_C \|\mathbf{u}_1\|_2^2$ for some width-independent constant $C$.*

*Proof.* **Correct scaling of updates (and learning rate) can be achieved by correctly scaling the spectral norm of the updates of weight matrices ($\sigma_B$ and $\sigma_C$).**

To give concrete update/learning rate scaling rules, we need to choose a concrete update rule. We first consider SGD as an example. First, we show that while spectral scaling of weight matrices does

not imply the correct scaling of (pre-)activations, spectral scaling conditions on the updates of weight matrices imply the correct scaling of the activation updates.

To simplify cumbersome notation, lets first consider the scale of updates $\mathbf{x}_1 = \mathbf{B}'\mathbf{u}_1$ after a single step of SGD on $((\mathbf{u}_1, y_1))$.

$$\left\|\Delta\mathbf{B}'\mathbf{u}_1\right\|_2 = \left\|\Lambda\Delta\mathbf{B}\mathbf{u}_1\right\|_2 = \left\|\Lambda\eta\bar{\mathbf{x}}_1\mathbf{u}_1^T\,\mathbf{u}_1\right\|_2 = |\eta|\,\|\mathbf{u}_1\|_2^2\,\|\Lambda\bar{\mathbf{x}}_1\|_2$$

Since the update matrix $\Delta\mathbf{B}'$ is a rank one matrix, we have

$$\left\|\Delta\mathbf{B}'\right\|_* = \left\|\Lambda\Delta\mathbf{B}\right\|_* = \eta\left\|(\Lambda\bar{\mathbf{x}}_1)\mathbf{u}_1^T\right\|_* = |\eta|\,\|\mathbf{u}_1\|_2\,\|\Lambda\bar{\mathbf{x}}_1\|_2$$

The updates $\Delta\mathbf{x}_1$ are given by $\Delta\mathbf{x}_1 = (\mathbf{B}' + \Delta\mathbf{B}')(\mathbf{u}_1 + \Delta\mathbf{u}_1) - \mathbf{B}'\mathbf{u}_1 = \mathbf{B}'\Delta\mathbf{u}_1 + \Delta\mathbf{B}'\mathbf{u}_1 + \Delta\mathbf{B}'\Delta\mathbf{u}_1$.

If the spectral norm of $\Delta\mathbf{B}'$ is scaled as $\Theta(\sqrt{\frac{N_x}{N_u}})$, then $\Delta\mathbf{B}'\mathbf{u}_1 \in \Theta(\sqrt{N_x})$. We also have that $\Delta\mathbf{B}'\Delta\mathbf{u}_1 \leq \|\Delta\mathbf{B}'\|_*\,\|\Delta\mathbf{u}_1\|_2 \in O(\sqrt{N_x})$. Since $\|\Delta\mathbf{u}_1\|_2 \in \Theta(\sqrt{N_u})$, when $\sigma_B$ is set to $\Theta(\sqrt{\frac{N_x}{N_u}})$, $\mathbf{B}'\Delta\mathbf{u}_1 \in \Theta(\sqrt{N_x})$. Therefore, unless the scale of $\Delta\mathbf{B}'\mathbf{u}_1$ perfectly cancels out with the remaining two terms, $\|\Delta\mathbf{x}_1\| \in \Theta(\sqrt{N_x})$.

Generalizing to more gradient steps (and $l > 1$) follows immediately if we assume that updates do not perfectly cancel out initial quantities, combined with the fact that $\mathbf{A}'$ does not change the width scaling. $\qquad\square$

# D  Additional Experiments

We conducted additional experiments to validate our theoretical derivations and to compare with baseline parameterizations for SSMs. Focusing exclusively on the SSM components, as parameterization for other parts of the model are already addressed in Yang and Hu (2021), Yang et al. (2023a), we decoupled the learning rates for SSM and non-SSM layers. First, we tuned the learning rate for the non-SSM layers and then evaluated test performance across various SSM learning rates, using the optimal non-SSM learning rate. The results, presented in Figure 4, show that the $\mu$P-SSM parameterization exhibits better monotonicity and stability.

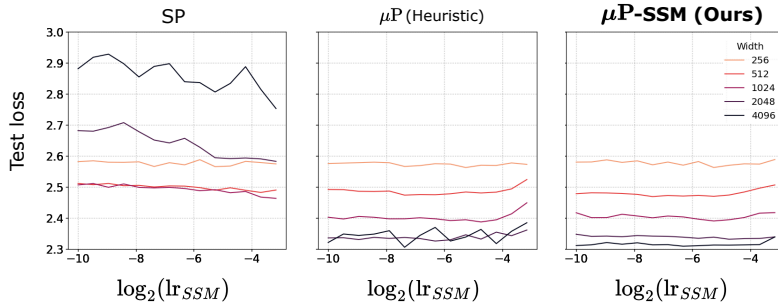

Figure 4: **Decoupled learning rates**. Test loss against SSM learning rate in Mamba with varying widths ($N_u$ and $N_x$) on the 4M tokens sampled from the WikiText-103 dataset. The learning rate for non-SSM components is fixed and chosen via hyper-parameter tuning.

In Figure 5, we present results on a randomly sampled subset of the Fineweb dataset. While computational constraints prevented us from training on the entire dataset or using larger model widths, at small scales, our observations on Fineweb align with the wikitext-103 results reported in the main paper.

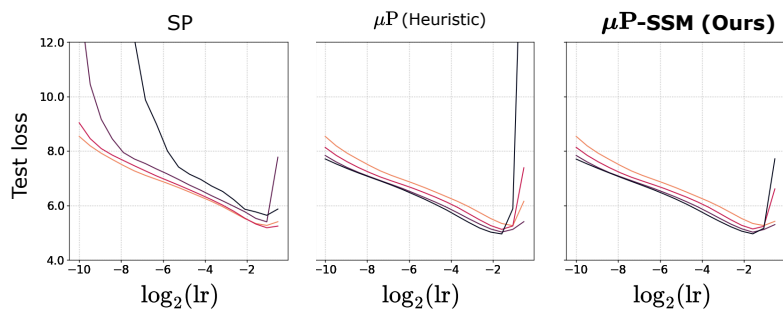

Figure 5: **Results on the FineWeb dataset**. Test loss against SSM learning rate in Mamba with varying widths ($N_u$ and $N_x$) on the 20M tokens from the FineWeb dataset.

